# Stochastic Optimal Control and Estimation with Multiplicative and Internal Noise

**Francesco Damiani**
Center for Brain and Cognition,
Department of Engineering
Pompeu Fabra University
Barcelona, ES
francesco.damiani@upf.edu

**Akiyuki Anzai**
Department of Brain and Cognitive Sciences
University of Rochester
Rochester, USA
aanzai@ur.rochester.edu

**Jan Drugowitsch**
Department of Neurobiology
Harvard Medical School
Boston, USA
jan_drugowitsch@hms.harvard.edu

**Gregory C. DeAngelis**
Department of Brain and Cognitive Sciences
University of Rochester
Rochester, USA
gdeangelis@ur.rochester.edu

**Rubén Moreno-Bote**
Center for Brain and Cognition, Department of Engineering,
Serra Húnter Fellow Programme
Pompeu Fabra University
Barcelona, ES
ruben.moreno@upf.edu

## Abstract

A pivotal brain computation relies on the ability to sustain perception-action loops. Stochastic optimal control theory offers a mathematical framework to explain these processes at the algorithmic level through optimality principles. However, incorporating a realistic noise model of the sensorimotor system — accounting for multiplicative noise in feedback and motor output, as well as internal noise in estimation — makes the problem challenging. Currently, the algorithm that is commonly used is the one proposed in the seminal study in [1]. After discovering some pitfalls in the original derivation, i.e., unbiased estimation does not hold, we improve the algorithm by proposing an efficient gradient descent-based optimization that minimizes the cost-to-go while only imposing linearity of the control law. The optimal solution is obtained by iteratively propagating in closed form the sufficient statistics to compute the expected cost and then minimizing this cost with respect to the filter and control gains. We demonstrate that this approach results in a significantly lower overall cost than current state-of-the-art solutions, particularly in the presence of internal noise, though the improvement is present in other circumstances as well, with theoretical explanations for this enhanced performance. Providing the optimal control law is key for inverse control inference, especially in explaining behavioral data under rationality assumptions.

## 1 Introduction

The sensorimotor system possesses a remarkable ability to reliably execute actions aligned with external and internal goals in spite of the noise sources affecting it [2, 3] and the numerous solutions

through which the same goals can be achieved [4]. Our nervous system is able to combine the complexity of the mechanical properties of the body with a regulatory control system [5]. How such control is implemented at a computational and algorithmic level is still an open question in systems neuroscience.

Optimal feedback control provides a valuable framework for understanding how the motor system creates coordinated and adaptable behavior [6, 5]. However, using optimality principles to infer the underlying computation [7] is a powerful yet potentially risky approach. Indeed, multiple independent factors can lead to discrepancies with experimental data. For instance, predictions might be based on unsuitable approximations or conditions. Consequently, although optimal feedback control, and more broadly, stochastic optimal control theory, can be regarded as ideal candidates to understand the principles of motor control [5, 6, 1, 7], their effectiveness depends on the mathematical correctness of the derived predictions and the accuracy of their assumptions.

Solving an optimal control problem implies deriving an optimal state-to-action policy, or control-law, to minimize a certain cost function, usually embedding control effort and task related goals [6]. The classical framework in which analytical solutions for optimal feedback control of stochastic, partially observable, continuous, non stationary, and high-dimensional systems can be derived, is the Linear-Quadratic-Additive-Gaussian (LQAG) problem (see Section 2), that assumes linear dynamics, a quadratic cost function and additive Gaussian noise [8, 9]. Despite having been used in the past to model motor control [10, 11, 12], these assumptions are too limiting to explain a wide range of observed, relevant behaviors, like smooth velocity profiles [1, 13, 14], speed-accuracy trade-offs [1, 15, 16, 17] and movement corrections [18, 3, 19]. Including a realistic noise model for the sensorimotor system is crucial to fill this gap, even at the cost of decreasing the mathematical tractability of the problem [1]. Indeed, accounting for control [1, 15, 6, 17, 20] and signal-dependent [1, 21, 22, 23, 14] noise at the motor output and sensory feedback level, and for internal noise [1, 2, 24] in the estimation process, permits explaining a broad range of experimentally observed phenomena [18, 3, 1, 25, 26], as discussed in Section 2.

The seminal study in [1], widely regarded as state-of-the-art for solving optimal control problems under this extended noise model, offers an iterative algorithm whereby a stochastic optimal control problem, incorporating multiplicative motor and sensory noise and additive internal noise, can be efficiently solved. Such an algorithm is currently used to explain behavioral data in the context of inverse optimal control [27]. Unfortunately, the derivation used in [1] erroneously assumes unbiased estimators. We propose an alternative algorithm that addresses this issue by assuming only linear control. The algorithm leverages the fact that the cost function can be computed from closed-form moment expressions, which can then be minimized numerically. To handle potential high computational costs, we derive an analytical counterpart for the optimization, based on the efficient propagation of cost function derivatives over time. For simplicity, the algorithm is derived for a simpler, yet relevant, case as outlined in Section 3.3, with extensions to the more general case also discussed. Our algorithm outperforms the solutions in [1] under internal noise, providing both theoretical and heuristic explanations for the performance differences. In a sensorimotor hand-reaching task, it reduces the cost by up to $90\%$ when internal noise constitutes $10\%$ of the total. This reveals qualitatively different behaviors, underscoring the importance of using the actual optimal controller, particularly when explaining behavior in a principled way [27].

In Section 2, to fix notation and ideas, we begin by formalizing the optimal control problem using the classic LQAG framework [8], addressing partial observability and assuming fully *additive* noise (Section 2.1). We then introduce the Linear-Quadratic-Multiplicative-Gaussian (LQMG) framework, which extends the noise model to include *multiplicative* noise in both control and observations, as well as additive *internal* noise, following the approach of [1] (Section 2.2). We demonstrate that the well-established solution from [1] produces suboptimal solutions in the presence of internal noise and prior to full algorithmic convergence (Section 2.3). In Section 3, we introduce a novel numerical algorithm that achieves optimal solutions and outperforms the approach in [1], as demonstrated empirically in Section 3.2. Finally, in Section 3.3, we present the analytical counterpart to the numerical algorithm.

## 2 Control and Estimation with Multiplicative and Internal Noise

### 2.1 The Classic Linear-Quadratic-Additive-Gaussian (LQAG) Problem

Stochastic optimal control theory formalizes the idea of controlling a dynamical system under partial observability to accomplish a goal [8]. In the LQAG problem (typically referred to as LQG), a linear system with latent state $x_t \in \mathbb{R}^m$

$$x_{t+1} = Ax_t + Bu_t + \xi_t \tag{1}$$

is controlled by a control signal $u_t \in \mathbb{R}^p$, with time-independent matrices $A \in \mathbb{R}^{m \times m}$ and $B \in \mathbb{R}^{m \times p}$, and initial condition $x_1$ – considering time-dependent matrices is straightforward. The term $\xi_t \in \mathbb{R}^m$ stands for a Gaussian random variable with zero mean and covariance $\Omega_\xi$ (we always consider i.i.d. random variables, but note that temporally correlated random variables can be generated by filtering the noise with the linear dynamics in Eq. 1). In the most relevant case, the controller does not have full access to the latent state $x_t$: the observation $y_t \in \mathbb{R}^k$ is a noisy version of $x_t$,

$$y_t = Hx_t + \omega_t , \tag{2}$$

with observation matrix $H \in \mathbb{R}^{k \times m}$ and $\omega_t \in \mathbb{R}^k$ being a Gaussian random variable with zero mean and covariance $\Omega_\omega$. Note that all noise sources are *additive*, that is, state independent, hence we refer to the classic LQG problem as LQAG. The controller $u_t \equiv u_t(y_{\tilde{t}})$ is constrained to be a function of the past observations only, $y_{\tilde{t}} = (y_1, ..., y_{t-1})$, and it must be optimized to minimize the total quadratic cost

$$\mathbb{E}[J] = \sum_{t=1}^{T} \mathbb{E}[j_t] = \sum_{t=1}^{T} \mathbb{E}\left[x_t^\intercal Q_t x_t + u_t^\intercal R_t u_t\right] , \tag{3}$$

where $T$ is the duration of the task, and $j_t$ is the cost per step in a trial, which includes a control cost (reflecting the internal goal of minimizing control effort) determined by the symmetric positive definite matrix $R_t \in \mathbb{R}^{p \times p}$, with $R_t > 0$, and a state cost (modeling potential external goals, such as minimizing the distance to a chosen target), determined by $Q_t \in \mathbb{R}^{m \times m}$. Again, $Q_t$ is symmetric and positive definite, $Q_t > 0$, and modulates the cost of the state being far from a chosen target. $J$ is the total cost, over a whole trial, while $\mathbb{E}[J]$ is the total expected cost. Here, the expectation $\mathbb{E}[f(\cdot)]$ denotes an average over all noise random variables with the same initial condition, that is, $\mathbb{E}[f(\cdot)] = \int dx_{2,...,T} \, dy_{1,...,T} f(\cdot) \, p(x_{2,...,T}, y_{1,...,T})$.

The optimal controller can be derived analytically [8] (see Appendix A.1). In summary, it is a linear function of the state estimate $\hat{x}_t$, $u_t = L_t \hat{x}_t$, where $L_t \in \mathbb{R}^{p \times m}$ is the control gain, and $\hat{x}_t$ is the estimator of the unobserved variable $x_t$, recursively computed with a linear Kalman filter

$$\hat{x}_{t+1} = A\hat{x}_t + Bu_t + K_t(y_t - H\hat{x}_t) \tag{4}$$

with filter gains $K_t \in \mathbb{R}^{m \times k}$ and initial condition $\hat{x}_1 = \mathbb{E}[x_1]$ ($x_1$ is a random Gaussian variable with covariance $\Sigma_{x_1}$) – to start with an unbiased estimate of the latent variable. Intuitively, the estimate at time $t + 1$ consists of a next-state prediction term (the first two terms in the r.h.s.) from the current estimate $\hat{x}_t$ plus a correction (third term) that depends on the prediction error, the difference between the new observation $y_t$ and the previous state prediction, weighted by its reliability. For example, if the noise magnitude is very large, $K_t = 0$, indicating that an open-loop strategy would be optimal [8].

For the classic LQAG problem, it is well known that the optimal Kalman filter satisfies the *orthogonality principle* [8], stating that the estimation error is orthogonal to the optimal estimator $\hat{x}_t$, i.e. $\mathbb{E}[(x_t - \hat{x}_t)\hat{x}_t^\intercal] = 0$, where we define from here onwards the expectation as $\mathbb{E}[f(\cdot)] = \int dx_{2,...,T} \, d\hat{x}_{2,...,T} f(\cdot) \, p(x_{2,...,T}, \hat{x}_{2,...,T})$, where $p$ is the joint density of latent and estimation variables with initial condition $\hat{x}_1 = \mathbb{E}[x_1]$ – e.g., $\mathbb{E}[\hat{x}_t] = \int d\hat{x}_t \hat{x}_t p(\hat{x}_t)$.

Also, as it is clear from the analytical expression (Appendix A.1), the computation for the optimal controller and filter gains are mathematically independent of each other, the so-called *separation principle* [28, 29, 30], which is closely related to the concept of certainty equivalence [31].

### 2.2 An Extended Noise Model: Optimal Control Beyond the LQAG Framework

Purely additive noise sources alone are insufficient to model the sensorimotor action-perception loop, as multiplicative noise affects both motor control [1, 15, 17, 16] and sensory feedback, including

visual and proprioceptive signals [6, 1, 23, 21, 22, 14]. For instance, stronger muscle forces produce greater noise [15, 17], and visual sensory noise increases in the periphery relative to the fovea [1, 23, 21, 22]. Accounting for these characteristics is crucial for explaining and reproducing various behavioral features in reaching movements, such as stereotyped bell-shaped velocity profiles [1, 13, 14] and the speed-accuracy trade-off [1, 15, 16, 17].

These considerations result in the Linear-Quadratic-Multiplicative-Gaussian (LQMG) model, with the following dynamics for state and sensory feedback [1]

$$x_{t+1} = Ax_t + Bu_t + \xi_t + \sum_{i=1}^{c} \varepsilon_t^i C_i u_t \tag{5}$$

$$y_t = Hx_t + \omega_t + \sum_{i=1}^{d} \rho_t^i D_i x_t . \tag{6}$$

In comparison to Eqs. 1-2 for the classic LQAG model, the LQMG model adds the final terms to account for multiplicative motor noise (Eq. 5) and sensory noise (Eq. 6). Specifically, performing a control action $u_t$ introduces noise proportional to the control signal itself (Eq. 5), while perceiving the state variable $x_t$ induces noise proportional to the observed state (Eq. 6). Here, $C_i \in \mathbb{R}^{m \times p}$ and $D_i \in \mathbb{R}^{k \times m}$ are constant scaling matrices, and $\varepsilon_t \in \mathbb{R}^c$ and $\rho_t \in \mathbb{R}^d$ are zero-mean Gaussian noise terms with covariances $\Omega_\varepsilon = \mathbb{I}$ and $\Omega_\rho = \mathbb{I}$, respectively [1]. For simplicity, in the expressions derived below, we set $c = d = 1$ to improve readability, without loss of generality.

As in the LQAG problem, the objective is to find the optimal control signal $u_t \in \mathbb{R}^p$ that depends solely on past observations $y_{1,\cdots,t-1}$ to minimize the cost function defined in Eq. 3. In this case, the optimal state estimate and corresponding filters are state-dependent and in general intractable, but we can simplify the problem by assuming, as in [1], that the filter is non-adaptive (i.e. independent of the state estimate, [1]), similar to the classic LQAG problem. This leads to the assumption that the state estimate follows the equation

$$\hat{x}_{t+1} = A\hat{x}_t + Bu_t + K_t(y_t - H\hat{x}_t) + \eta_t , \tag{7}$$

with the same terminology as in Eq. 4. The initial state $x_1$ and its estimate $\hat{x}_1$, assumed to be independent, are Gaussian variables with the same mean $\mathbb{E}[x_1] = \mathbb{E}[\hat{x}_1]$, and covariances $\Sigma_{x_1}$ and $\Sigma_{\hat{x}_1}$, respectively. This dynamics of the state estimate only differs from Eq. 7 due to the presence of an additional zero-mean Gaussian noise term $\eta_t \in \mathbb{R}^m$ with covariance $\Omega_\eta$, which models the possibility of inefficient filtering of the past observations. This noise term may represent internal fluctuations in neural activity [2, 24, 32, 3] or inaccuracies in the filtering process, and is important for explaining behavioral data [1].

Under the new LQMG model (Eqs. 5-7), the task is to find the optimal control signal $u_t = u_t(\hat{x}_t)$, for $t = 1, ..., T - 1$, and the filter gains $K_{1,\cdots,T-2}$, that minimize the quadratic cost in Eq. 3.

## 2.3 State-of-the-Art Solutions for the LQMG Model: Causes of Suboptimality

It is important to first recognize that solving the LQMG problem in Eqs. 5-7 is significantly more complex than in the classic LQAG problem: while in the latter the separation principle applies, allowing for a direct analytical solution (Appendix A.1), in the former the principle does not hold, resulting in tightly intertwined controller and filter gains. Notably, the presence of internal noise alone introduces control-estimation interdependencies— a factor previously overlooked in earlier approaches.

The algorithm currently used to solve the optimal control problem under the LQMG model (Eqs. 5-7) is the one introduced in the seminal work of [1], whose solutions are detailed in Appendix A.2. The impact of this research extends beyond theoretical considerations [27, 33, 34, 35, 36, 37, 38]. We now describe some pitfalls in the original derivation and explain why certain assumptions fail, leading to suboptimality. In Section 3, we propose an alternative algorithm, and in Section 3.2, we demonstrate that our solutions outperform the previous ones.

The algorithm in [1] assumes, throughout the derivation of the optimal control-estimation loop, that the estimator is unbiased, meaning $\mathbb{E}[x_t|\hat{x}_t] = \hat{x}_t$. However, this condition is never truly satisfied. To illustrate this conceptually, we can consider a one-dimensional toy problem involving a partially observable stochastic process $x_t$ (Fig. 1a). Assume that at time $t - 1$, the condition

$\mathbb{E}[x_{t-1}|\hat{x}_{t-1}] = \hat{x}_{t-1}$ holds. Now, suppose that at the same time, a large positive fluctuation, possibly caused by sensory or internal noise, affects the agent's internal estimate. As a result, while the actual state $x_t$ changes only slightly compared to $x_{t-1}$, the state estimate $\hat{x}_t$ changes significantly, so that $\hat{x}_t \gg \hat{x}_{t-1}$. At this point, it becomes clear that the expected value of $x_t$ conditioned on $\hat{x}_t$ cannot equal $\hat{x}_t$, thus violating the unbiasedness condition (see Fig. 1a). This effect is more pronounced when the state estimate undergoes large fluctuations, but a similar bias, although smaller, would still be present with minor fluctuations.

We demonstrate this issue numerically by considering a one-dimensional problem ($m = p = k = 1$) with multiplicative, additive, and internal noise (for the details see Appendix A.3). In the absence of internal noise, the violation of unbiasedness is still present, though it becomes pronounced only for large values of $\hat{x}_t$ (Fig. 1b,c). However, when internal noise is introduced, the bias increases substantially since the internal fluctuations are not directly attenuated by the gains $K_t$ (see also Appendix A.3).

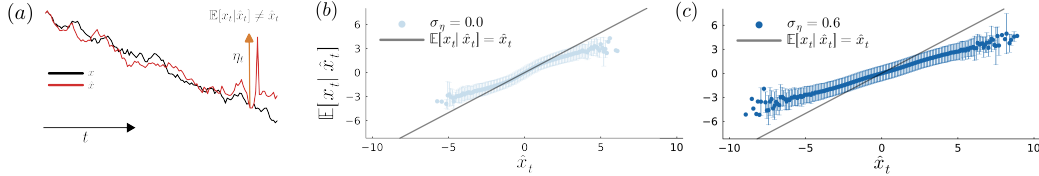

Figure 1: *The invalidity of the unbiasedness condition*. **(a)** A toy example illustrating estimation bias. The black line represents the dynamics of a partially observable process, $x_t$, while the red line shows the state estimate, $\hat{x}_t$, biased by random internal fluctuations from the noise term $\eta_t$ (orange arrow). **(b-c)** $\mathbb{E}[x_t|\hat{x}_t]$, for $t = 8$, as a function of $\hat{x}_t$ for $\sigma_\eta = 0.0, 0.6$, respectively, using the solutions from [1]. The conditional expectation $\mathbb{E}[x_t|\hat{x}_t]$ is computed through Monte Carlo simulations (dots $\pm$ error bars: mean $\pm$ 1std). The gray straight line represents the identity line, where $\mathbb{E}[x_t|\hat{x}_t] = \hat{x}_t$.

As said above, in the classic LQAG problem the optimal Kalman filter satisfies the orthogonality principle [8]. In contrast, when internal noise is non-zero we show that the orthogonality principle does not hold anymore for the optimal filter (see Appendix A.3.1). We further show that the condition of unbiasedness implies the orthogonality principle, but not the other way around. Therefore, applying unbiasedness to solve the optimal control problem introduced in Section 2 leads to suboptimal solutions when internal noise is present, irrespective of control or signal-dependent noise. A similar issue arises when the algorithm has not yet converged, even for zero internal noise, because the orthogonality principle only holds for optimized filter gains (and in the absence of internal noise). As a result, the algorithm in [1] also fails to produce the optimal control when derived with a fixed suboptimal estimator, and to produce the optimal filter estimate with a fixed suboptimal controller.

# 3   A Novel Algorithm for Optimal Control Problems

To address the issues outlined in the previous section, we introduce an alternative method for solving the LQMG problem presented in Section 2.2. We compute the expected total accumulated cost, $\mathbb{E}[J]$, by averaging over all stochastic terms present in Eqs. 5-7, and as a function of $L_t$ and $K_t$. For fixed $L_t$ and $K_t$, $\mathbb{E}[J]$ serves as the objective function for a standard gradient descent algorithm aimed at minimizing it. In Section 3.1, we detail the computation of this objective function through moment propagation and discuss the minimization process with respect to $L_{1,\cdots,T-1}$ and $K_{1,\cdots,T-2}$. In Section 3.2, we demonstrate that this approach outperforms state-of-the-art algorithms. In Section 3.3, we derive the analytical counterpart to our numerical algorithm.

## 3.1   Minimization of Theoretical Expected Cost Through Numerical Gradient Descent (GD)

Our method assumes linear control, where the control signal $u_t$ is linear in the internal estimate

$$u_t = L_t \hat{x}_t . \tag{8}$$

This assumption is not very limiting, as it is correct for the classic LQAG problem and has been used before for the LQMG problem [1]. Crucially, we do not assume unbiasedness to solve the

optimal control problem. The expected total accumulated cost is computed by propagating the first two moments of $x$ and $\hat{x}$ in closed form. Given that both control and estimation are linear in $x$ and $\hat{x}$, and the cost function is quadratic in $x$ and $u$, the first and second moments act as sufficient statistics. As a result, no additional approximations (e.g., assuming Gaussianity of $x$ and $\hat{x}$) are required to find the optimal solutions. By using Eq. 8 and the formula for the expected cost of a quadratic form, Eq. 3 can be rewritten as

$$\mathbb{E}[J] = \sum_{t=1}^{T} \mathbb{E}[j_t] = \sum_{t=1}^{T} (\mathbb{E}[x_t]^\intercal Q_t \mathbb{E}[x_t] + \mathbb{E}[\hat{x}_t]^\intercal L_t^\intercal R_t L_t \mathbb{E}[\hat{x}_t] + $$
$$+ Tr[Q_t \Sigma_{x_t}] + Tr[L_t^\intercal R_t L_t \Sigma_{\hat{x}_t}]), \tag{9}$$

where $Tr[\cdot]$ stands for the trace operation, $\Sigma_{x_t}$ is the covariance matrix of the latent state $x_t$ and $\Sigma_{\hat{x}_t}$ is the covariance of the state estimate at time $t$. Note that $\mathbb{E}[x_t], \mathbb{E}[\hat{x}_t], \Sigma_{x_t}$ and $\Sigma_{\hat{x}_t}$ will implicitly depend on $L_{1,...,t-1}$ and $K_{1,...,t-1}$. From Eqs. 5-7 we can derive the update equations to propagate the first and second-order moments $\mathbb{E}[x_t]$ $\mathbb{E}[\hat{x}_t]$, $\Sigma_{x_t}$ and $\Sigma_{\hat{x}_t}$ in a closed-form manner, in order to compute the total expected cost $\mathbb{E}[J]$ at fixed $L_{1,\cdots,T-1}$ and $K_{1,\cdots,T-2}$ (for the derivation see Appendix A.4.1). Here and in the following we set $c = d = 1$ for simplicity (the case $c, d > 1$ follows simply by replacing terms with $D$ or $C$ matrices by $C_i$ and $D_i$, respectively, and sum over $i$). To rewrite our results in a more compact form, we define (similarly to [27, 39])

$$\mu_t = \begin{pmatrix} \mu_{x_t} \\ \mu_{\hat{x}_t} \end{pmatrix} = \begin{pmatrix} \mathbb{E}[x_t] \\ \mathbb{E}[\hat{x}_t] \end{pmatrix}, \tag{10}$$

$$\Sigma_t = \begin{pmatrix} \Sigma_{x_t} & \Sigma_{x_t, \hat{x}_t} \\ \Sigma_{\hat{x}_t, x_t} & \Sigma_{\hat{x}_t} \end{pmatrix}, \tag{11}$$

$$M_t = \begin{pmatrix} A & BL_t \\ K_t H & A + BL_t - K_t H \end{pmatrix} \tag{12}$$

and

$$G_t = \begin{pmatrix} CL_t(\Sigma_{\hat{x}_t} + \mu_{\hat{x}_t}\mu_{\hat{x}_t}^\intercal)L_t^\intercal C^\intercal + \Omega_\xi & 0 \\ 0 & K_t D(\Sigma_{x_t} + \mu_{x_t}\mu_{x_t}^\intercal)D^\intercal K_t^\intercal + K_t \Omega_\omega K_t^\intercal + \Omega_\eta \end{pmatrix}, \tag{13}$$

where $\Sigma_{x_t, \hat{x}_t} = \mathbb{E}[x_t \hat{x}_t^\intercal] - \mathbb{E}[x_t]\mathbb{E}[\hat{x}_t]^\intercal$ and $\Sigma_{\hat{x}_t, x_t} = \Sigma_{x_t, \hat{x}_t}^\intercal$. We have defined $\mu_t$ as a column vector whose block elements are $m-$dimensional vectors. Similarly, $\Sigma_t$, $M_t$ and $G_t$ are block matrices, whose block elements are $m \times m$ matrices.

With these definitions, we see that the first and second moments propagate in a closed manner as

$$\mu_{t+1} = M_t \mu_t, \tag{14}$$

$$\Sigma_{t+1} = M_t \Sigma_t M_t^\intercal + G_t. \tag{15}$$

In other words, if the first and second moments are known at time $t$, their values can be recursively computed at time $t + 1$, and no other moments are involved in the calculations.

As a result, given the initial conditions for $\mu_1$ and $\Sigma_1$, we can compute the expected accumulated cost $\mathbb{E}[J]$ at fixed $L_{1,\cdots,T-1}$ and $K_{1,\cdots,T-2}$, by using Eqs. 14-15 together with Eq. 9. The pseudo-code for the algorithm to compute the expected cost $\mathbb{E}[J]$ is provided in Appendix A.4.2, Algorithm 1. To find the optimal control and filter gains we would then use $\mathbb{E}[J]$ as the objective function of a numerical gradient descent procedure. The analytical gradient descent counterpart is discussed in Section 3.3.

## 3.2 Experiments: Enhanced Performance with the GD Algorithm

We apply our algorithm and compare it with the state-of-the-art solutions in two scenarios governed by a linear dynamical system (Eqs. 5-8). Hereafter, GD refers to our numerical algorithm (Section 3.1), and TOD refers to the algorithm from [1]. First, in a simplified one-dimensional reaching task ($m = p = k = 1$) with all noise sources present, we show that for non-zero internal noise, $\Omega_\eta > 0$, GD outperforms TOD, resulting in a lower accumulated cost. Second, in a reaching task with a four-dimensional state and one-dimensional control and sensory feedback ($m = 4, p = k = 1$), GD predicts qualitatively different behavior and shows a 90% performance improvement when internal noise contributes 10% of the total.

**One-Dimensional Case: Understanding the Qualitative Differences**    We examine the case where $m = p = k = 1$, incorporating multiplicative, additive, and internal noise. The system parameters are provided in Table 2 in Appendix A.5.1 (note that we define the strength of the internal noise as $\sigma_\eta = \sqrt{\Omega_\eta}$). With non-zero internal noise, our algorithm achieves lower-cost solutions compared to the method proposed in [1] (Fig. 2a). This improved performance arises from different modulations of $L_t$ and $K_t$ as $\sigma_\eta$ varies (Figs. 2b, c). Crucially, our solution results in control gains that decrease as internal noise increases, while the TOD solution shows little sensitivity to internal noise magnitude (Fig. 2b): internal noise increasingly intertwines the optimal solutions for $K_t$ and $L_t$. In Appendix A.5.2, we provide a geometric interpretation of why this modulation is optimal, demonstrating that this optimality enhances adaptability, and showing how internal noise disrupts the orthogonality principle.

As outlined in Section 3, the incorrect unbiasedness condition implies the orthogonality principle. Thus, even if the estimator is biased, the algorithm in [1] finds the optimal solution with zero internal noise, as this principle holds for the optimal Kalman filter. However, for non-zero internal noise, $\sigma_\eta > 0$, the TOD algorithm underperforms due to the breakdown of the orthogonality principle. In Appendix A.5.3, we also discuss that this suboptimality is observed before the algorithm converges, when the optimal control law is derived from fixed suboptimal filters, and vice versa, regardless of the presence of internal noise.

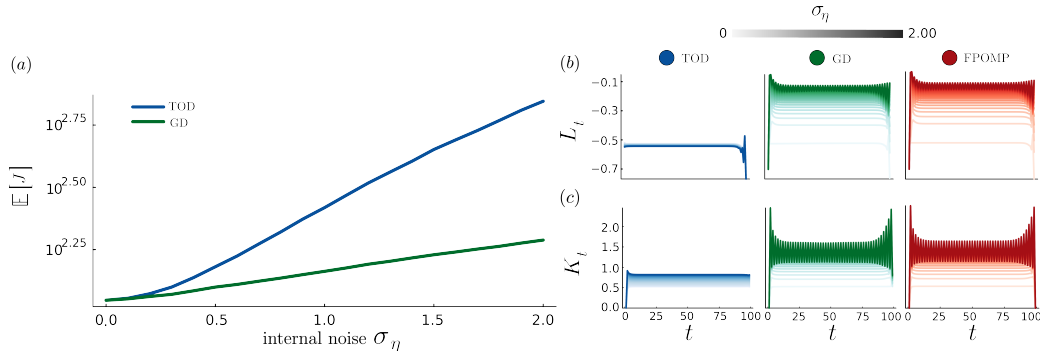

Figure 2: *Enhanced performance and different solutions with internal noise.* **(a)** Expected accumulated cost $\mathbb{E}[J]$, computed by averaging the quantity from Eq. 3 over $50k$ trials, as a function of the internal noise strength $\sigma_\eta$, for TOD [1] and GD (Section 3.1) algorithms (mean $\pm$ 1SEM from Monte Carlo simulations, error bars not visible as too small). The expected value aligns with our theoretical estimate of $\mathbb{E}[J]$, as derived in Section 3.1. **(b-c)** Optimal control and filter gains, $L_t$ and $K_t$, for TOD and GD and algorithms. We also present the solutions derived from the analytical counterpart of the numerical GD algorithm to demonstrate that they match the optimal solutions ('Fixed Point Optimization with Moments Propagation' – FPOMP – algorithm, see Section 3.3).

**Multi-Dimensional Case: A Motor Control Application**    To demonstrate the scalability of our algorithm to more realistic motor control scenarios, we examine a problem with a four-dimensional state vector ($m = 4$) and one-dimensional control and sensory feedback ($p = k = 1$). The task is identical to that in [1], except that we include internal noise, which was absent in the original formulation. We model a single-joint reaching movement aimed at minimizing the distance between the hand position $p_t$ and a target, while minimizing control effort. The state variable of the problem is $x_t = [p_t, \dot{p}_t \equiv dp_t/dt, f_t, g_t]$, where $f_t$ is the force acting on the hand and $g_t$ is an auxiliary variable used to filter the control signal $u_t$ (see [1] and [40] for a more detailed discussion). We include control and state-dependent noise, as well as internal noise, perturbing the estimate of $p_t$. Note that now we denote $\sigma_\eta = \sqrt{\Omega_\eta^{1,1}}$. All parameters are listed in Appendix A.5.4.

Our results confirm the findings from the previous scenario in this more complex sensorimotor task. The GD algorithm achieves a lower expected accumulated cost, with the performance gap widening as internal noise $\sigma_\eta$ increases (Fig. 3a). This is achieved by reducing control gains with increasing $\sigma_\eta$ (Fig. 3b), resulting in a smoother control signal on individual trials (Fig. 3c) and overall reduced control effort (Fig. 3d). These adjustments lead to two key behavioral outcomes: slower movements compared to TOD solutions and significantly reduced trial-to-trial variability (Fig. 3e). As mentioned

earlier, GD outperforms the algorithm proposed by [1], reducing the cost by up to $90\%$ when internal noise contributes approximately $10\%$ of the total noise ($\sigma_\eta = 0.05$). To further quantify the impact of internal noise in this scenario and enhance clarity, we calculate the ratio between the average fluctuation amplitude of the state estimate (*FA*, the standard deviation of the state estimate) and the average range of variation of the state (*RV*, the range of variation in position $p_t$, defined as $\max_t(p_t) - \min_t(p_t)$). The resulting ratio is *FA/RV* $\approx 0.5$ for $\sigma_\eta = 0.05$ (see also Appendix A.5.5).

We emphasize that the GD algorithm naturally handles arbitrarily high-dimensional problems without requiring any further adjustments. Algorithm 1 in Appendix A.4.2, which serves as the objective function for the numerical optimization via gradient descent, is designed to accommodate arbitrary dimensions for state, control, and sensory feedback, as well as trial duration. However, the time horizon must remain finite by assumption, similar to [1]. We empirically demonstrate the scalability of our algorithm by applying it to a significantly higher-dimensional problem, where the linear dynamical system is governed by random matrices with Gaussian entries. Specifically, we consider $m = 10$, $p = 4$, and $k = 10$ for the dimensions of state, control, and observation, respectively (Appendix A.5.6).

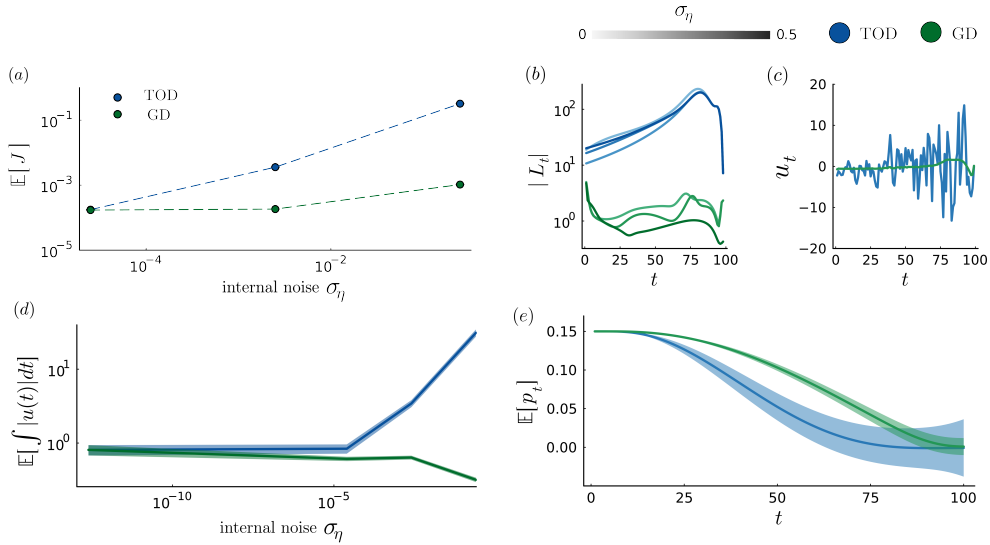

Figure 3: *Single-joint reaching task*. **(a)** Expected accumulated cost $\mathbb{E}[J]$ as a function of the internal noise (mean $\pm$ 1SEM, error bars not visible as too small), for TOD and GD algorithms. **(b)** Magnitude of the control gain vector as a function of time for TOD and GD solutions. **(c)** control signal $u_t$ in a sample trial for the two algorithms for $\sigma_\eta = 0.05$. **(d)** Amount of control as $\sigma_\eta$ increases, that is, mean integral of the absolute control signal for the two algorithms. **(e)** Mean position over time for the two solutions. All averages are over $50k$ trials; shadowed areas are SEM.

### 3.3 An Analytical Approach: FPOMP Algorithm

Although, as discussed in the previous section, our GD algorithm performs well for arbitrarily high-dimensional problems (see also Appendix A.5.6), its application to complex, real-world tasks can become computationally expensive. With $L_{1,\cdots,T-1} \in \mathbb{R}^{p \times m}$ and $K_{1,\cdots,T-2} \in \mathbb{R}^{m \times k}$, the total number of parameters to optimize via numerical Gradient Descent is $mp(T - 1) + mk(T - 2)$, which can become quite large for high-dimensional state spaces. For example, in the 1D problem, the TOD and GD algorithms have comparable and short computation times. However, for the multi-dimensional sensorimotor task presented above, the GD algorithm takes significantly longer: while the TOD algorithm completes in just a few minutes on a standard laptop, the GD optimization requires several hours (approximately 4 hours).

To address this, we propose an analytically derived iterative algorithm, where we alternate between finding the optimal (i.e., cost-minimizing) $L_{1,\cdots,T-1}$ and $K_{1,\cdots,T-2}$, denoted as $L_t^*$ and $K_t^*$, for fixed state and state estimate moments, $\mu_t$ and $\Sigma_t$, and re-computing these moments in light of the updated $L_t$'s and $K_t$'s. We refer to this method as the 'Fixed Point Optimization with Moments

Propagation' (FPOMP) algorithm. Note that, when optimizing for $L_t$ and $K_t$, we condition also on all the future control and filter gains, therefore not only on $\mu_t$ and $\Sigma_t$, but also on $L_{t+1,\cdots,T-1}$ and $K_{t+1,\cdots,T-2}$. To simplify the notation, we will omit this implicit dependence and explicitly state only the dependence on $\mu_t$ and $\Sigma_t$. Therefore, at each iteration, we identify the critical points of the total conditional expected cost with respect to $L_t$ and $K_t$. We can compute the conditional expected cost per step at time $t + i$, $i = 0, ..., T - t$ as

$$\mathbb{E}[j_{t+i}|\mu_t, \Sigma_t] = \mathbb{E}[x_{t+i}|\mu_t, \Sigma_t]^\intercal Q_{t+i}\mathbb{E}[x_{t+i}|\mu_t, \Sigma_t]+$$
$$+ \mathbb{E}[\hat{x}_{t+i}|\mu_t, \Sigma_t]^\intercal L_{t+i}^\intercal R_{t+i}L_{t+i}\mathbb{E}[\hat{x}_{t+i}|\mu_t, \Sigma_t]+ \qquad (16)$$
$$+ Tr[Q_{t+i}\Sigma_{x_{t+i}|\mu_t, \Sigma_t}] + Tr[L_{t+i}^\intercal R_{t+i}L_{t+i}\Sigma_{\hat{x}_{t+i}|\mu_t, \Sigma_t}] \,,$$

where $\mathbb{E}[x_{t+i}|\mu_t, \Sigma_t]$, $\mathbb{E}[x_{t+i}|\mu_t, \Sigma_t]$, $\Sigma_{x_{t+i}|\mu_t, \Sigma_t}$ and $\Sigma_{\hat{x}_{t+i}|\mu_t, \Sigma_t}$ are computed by propagating the moments $\mu_t$ and $\Sigma_t$ (Eqs. 14-15) until $\tilde{t} = t + i$. Indeed, as discussed in Section 3.1, $\mu_t$ and $\Sigma_t$ serve as sufficient statistics for computing the expected cost. To derive $L_t^*$ and $K_t^*$, we set the derivatives of the expected cost in Eq. 9 to zero. Excluding the constant terms, we obtain

$$\frac{\partial}{\partial L_t} \sum_{i=0}^{T-t} \mathbb{E}[j_{t+i}|\mu_t, \Sigma_t] = 0 \qquad (17)$$

$$\frac{\partial}{\partial K_t} \sum_{i=1}^{T-t} \mathbb{E}[j_{t+i}|\mu_t, \Sigma_t] = 0 \,. \qquad (18)$$

As shown in Appendix A.6.1 and A.6.2, solving Eqs. 17-18 leads to a backward algorithm to compute $L_t^*$ and $K_t^*$,

$$L_t^* = f(\mu_t, \Sigma_t, L_{t+1,\cdots,T-1}^*, K_{t+1,\cdots,T-2}^*) \qquad (19)$$
$$K_t^* = g(\mu_t, \Sigma_t, L_{t+1,\cdots,T-1}^*, K_{t+1,\cdots,T-2}^*) \,, \qquad (20)$$

with $t = 1, ..., T - 1$ for $L_t^*$ and $t = 1, ..., T - 2$ for $K_t^*$. From this we can build a recursive relationship that, starting from an initial guess for $L_{1,\cdots,T-1}^*$ and $K_{1,\cdots,T-2}^*$, iteratively computes all the moments $\mu_{1,\cdots,T}$ and $\Sigma_{1,\cdots,T}$ (Eqs. 14-15) at fixed $L_{1,\cdots,T-1}^*$ and $K_{1,\cdots,T-2}^*$. Given those moments, $L_{1,\cdots,T-1}^*$ and $K_{1,\cdots,T-2}^*$ are updated by using Eqs. 19-20, and so on, until convergence is attained. The pseudo-code for the FPOMP algorithm, with its implementation details, can be found in Appendix A.6.3, Algorithm 2. In such a way, we eliminate the numerical optimization procedure, making the algorithm suitable for extremely large optimal control problems. The FPOMP algorithm is flexible and works for arbitrary dimensions of state, control, sensory feedback, and trial duration, with computational costs and runtime comparable to those of the approach proposed in [1].

In Appendix A.6.1, we explicitly solve Eqs. 17-18 for the one-dimensional case, while in Appendix A.6.2 we extend the approach to a multi-dimensional scenario, considering, for the sake of simplicity, the LQAG problem (but, crucially, including internal noise), to prove the generalizability of Algorithm 2. In Appendix A.6.2, we provide the solution for Eq. 17, with the same procedure applying to Eq. 18. We also discuss the potential extension to the full noise model. Lastly, in Appendix A.7, we examine the assumption of linear dynamics and extend our approach to a switching linear dynamical system to make it less restrictive.

**Experiments** To empirically validate our iterative algorithm, we apply it to the same one-dimensional problem discussed in Section 3.2. The FPOMP algorithm aligns with the optimal solutions found by the GD algorithm, resulting in identical solution and performance (Figs. 2b,c). In Fig. 2a, the FPOMP algorithm follows the same cost trend as the GD algorithm, with the curves overlapping (the FPOMP curve is omitted for clarity, see also Fig. 10 in Appendix A.8.1).

In Appendix A.8.2, we analyze the same multi-dimensional task as in Section 3.2, excluding multiplicative noise but including internal noise. The results show that the FPOMP algorithm matches the GD optimal solutions for the controller and outperforms TOD when $\sigma_\eta > 0$. Even in its current form, FPOMP surpasses the method from [1] when internal noise is considered.

## 4 Conclusion

In this paper, we provide a novel approach for solving stochastic optimal control problems adapted to the noise characteristics of the human sensorimotor system. Our work builds on the seminal

study in [1], where the classical LQG framework (called here LQAG) is extended to the LQMG framework to include both control and signal-dependent noise, as well as internal noise in the estimation process. This extension provides a more realistic description of the sensorimotor system, enabling the reproduction of a larger sample of behavioral data in motor control, albeit at the cost of reduced mathematical tractability.

However, the solution derived in [1], which is widely used [27, 33, 41, 34, 35, 42, 36, 37, 38], suffers from an ill-conditioned derivation. Specifically, that solution assumes unbiased estimators –a condition that, as we prove numerically and conceptually in this work, does not hold, leading to suboptimal performance when internal noise is considered or before algorithmic convergence. This suboptimality arises from the close relationship between unbiasedness and the orthogonality principle of an optimal estimator, where the former, mathematically, implies the latter. Yet, the orthogonality principle is satisfied by the optimal filter only in the absence of internal noise and under algorithmic convergence.

Assuming only that control is linear in the current state estimate, we derive an alternative algorithm that optimizes control and estimation without requiring unbiasedness. The optimal solution is obtained by propagating sufficient statistics to compute the expected cost, which is minimized via numerical gradient descent on filter and control gains. For a more constrained, but still relevant, version of the problem, we derive the analytical counterpart, which alternates between forward propagation of moments and backward optimization of control and filter gains until convergence. This makes our approach suitable for high-dimensional problems, significantly reducing computational cost.

We demonstrate superior performance in the presence of internal noise and before convergence is reached (that is, when filter or control gains are fixed at suboptimal values, regardless of the level of internal noise), and provide both mathematical and heuristic explanations. Joint modulation of control and filter gains helps filter internal fluctuations, enhancing adaptability and generalization across internal noise levels, as discussed in Appendix A.5.2. By applying our algorithm to a sensorimotor task, we make novel behavioral predictions that distinguish our solution from previous ones. Specifically, we find that control gains decrease with increasing internal noise, leading to smoother control signals in individual trials. This results in slower movements with reduced trial-to-trial variability.

In summary, our algorithm extends optimal feedback control to a broader range of problems in systems neuroscience.

**Limitations and Future Work**    One limitation of our work is the assumption of state-independent filter gains for the optimal estimator: in the presence of multiplicative noise, non-adaptive estimation proves sub-optimal. Additionally, incorporating more realistic cost functions could extend our framework beyond the traditional quadratic dependence. Further investigation into the connections between our optimal control law and biologically plausible learning rules [43] may also be necessary. Moreover, we have not formally demonstrated the convergence properties of our algorithm to a global minimum, although our algorithm is guaranteed to converge at least to a local minima, and we did not find any numerical evidence for multiple local minima. The next immediate step is to derive the FPOMP algorithm for the general case with multiplicative noise, as discussed in Appendix A.6.2.

## Acknowledgments and Disclosure of Funding

This work was supported by: NEI-NIH R01 EY016178; Grant PRE2021-097778, funded by MI-CIU/AEI/10.13039/501100011033 and by "ESF+"; "Project PID2023-146524NB-I00 financed by MCIN/AEI/10.13039/501100011033/ ERDF, EU, the Spanish State Research Agency and the European Union, and ICREA Academia.

## References

[1] Emanuel Todorov. Stochastic optimal control and estimation methods adapted to the noise characteristics of the sensorimotor system. *Neural computation*, 17(5):1084–1108, 2005.

[2] A Aldo Faisal, Luc PJ Selen, and Daniel M Wolpert. Noise in the nervous system. *Nature reviews neuroscience*, 9(4):292–303, 2008.

[3] David W Franklin and Daniel M Wolpert. Computational mechanisms of sensorimotor control. *Neuron*, 72(3):425–442, 2011.

[4] Emmanuel Guigon, Pierre Baraduc, and Michel Desmurget. Computational motor control: redundancy and invariance. *Journal of neurophysiology*, 97(1):331–347, 2007.

[5] Stephen H Scott. Optimal feedback control and the neural basis of volitional motor control. *Nature Reviews Neuroscience*, 5(7):532–545, 2004.

[6] Emanuel Todorov and Michael I Jordan. Optimal feedback control as a theory of motor coordination. *Nature neuroscience*, 5(11):1226–1235, 2002.

[7] Emanuel Todorov. Optimality principles in sensorimotor control. *Nature neuroscience*, 7(9):907–915, 2004.

[8] Mark Davis. *Stochastic modelling and control*. Springer Science & Business Media, 2013.

[9] Robert F Stengel. *Optimal control and estimation*. Courier Corporation, 1994.

[10] Gerald E Loeb, WS Levine, and Jiping He. Understanding sensorimotor feedback through optimal control. In *Cold Spring Harbor symposia on quantitative biology*, volume 55, pages 791–803. Cold Spring Harbor Laboratory Press, 1990.

[11] Arthur D Kuo. An optimal control model for analyzing human postural balance. *IEEE transactions on biomedical engineering*, 42(1):87–101, 1995.

[12] Bruce Richard Hoff. *A computational description of the organization of human reaching and prehension*. University of Southern California, 1992.

[13] Tamar Flash and Neville Hogan. The coordination of arm movements: an experimentally confirmed mathematical model. *Journal of neuroscience*, 5(7):1688–1703, 1985.

[14] Christopher M Harris and Daniel M Wolpert. Signal-dependent noise determines motor planning. *Nature*, 394(6695):780–784, 1998.

[15] GG Sutton and K Sykes. The variation of hand tremor with force in healthy subjects. *The Journal of physiology*, 191(3):699–711, 1967.

[16] Emanuel Todorov. Cosine tuning minimizes motor errors. *Neural computation*, 14(6):1233–1260, 2002.

[17] Richard A Schmidt, Howard Zelaznik, Brian Hawkins, James S Frank, and John T Quinn Jr. Motor-output variability: a theory for the accuracy of rapid motor acts. *Psychological review*, 86(5):415, 1979.

[18] Reza Shadmehr and John W Krakauer. A computational neuroanatomy for motor control. *Experimental brain research*, 185:359–381, 2008.

[19] Joseph Y Nashed, Frédéric Crevecoeur, and Stephen H Scott. Influence of the behavioral goal and environmental obstacles on rapid feedback responses. *Journal of neurophysiology*, 108(4):999–1009, 2012.

[20] Kelvin E Jones, Antonia F de C Hamilton, and Daniel M Wolpert. Sources of signal-dependent noise during isometric force production. *Journal of neurophysiology*, 88(3):1533–1544, 2002.

[21] Chistina A Burbeck and Yen Lee Yap. Two mechanisms for localization? evidence for separation-dependent and separation-independent processing of position information. *Vision research*, 30(5):739–750, 1990.

[22] David Whitaker and Keziah Latham. Disentangling the role of spatial scale, separation and eccentricity in weber's law for position. *Vision research*, 37(5):515–524, 1997.

[23] Emanuel Vassilev Todrov. *Studies of goal directed movements*. PhD thesis, Massachusetts Institute of Technology, 1998.

[24] Rubén Moreno-Bote, Jeffrey Beck, Ingmar Kanitscheider, Xaq Pitkow, Peter Latham, and Alexandre Pouget. Information-limiting correlations. *Nature neuroscience*, 17(10):1410–1417, 2014.

[25] Philippe Vindras and Paolo Viviani. Frames of reference and control parameters in visuomanual pointing. *Journal of Experimental Psychology: Human Perception and Performance*, 24(2):569, 1998.

[26] Andrew M Gordon, Goran Westling, Kelly J Cole, and Roland S Johansson. Memory representations underlying motor commands used during manipulation of common and novel objects. *Journal of neurophysiology*, 69(6):1789–1796, 1993.

[27] Matthias Schultheis, Dominik Straub, and Constantin A Rothkopf. Inverse optimal control adapted to the noise characteristics of the human sensorimotor system. *Advances in Neural Information Processing Systems*, 34:9429–9442, 2021.

[28] Sanjoy K Mitter. Filtering and stochastic control: A historical perspective. *IEEE Control Systems Magazine*, 16(3):67–76, 1996.

[29] Tryphon T Georgiou and Anders Lindquist. The separation principle in stochastic control, redux. *IEEE Transactions on Automatic Control*, 58(10):2481–2494, 2013.

[30] D Peter Joseph and T Julius Tou. On linear control theory. *Transactions of the American Institute of Electrical Engineers, Part II: Applications and Industry*, 80(4):193–196, 1961.

[31] Henk Van de Water and Jan Willems. The certainty equivalence property in stochastic control theory. *IEEE Transactions on Automatic Control*, 26(5):1080–1087, 1981.

[32] Mark M Churchland, Afsheen Afshar, and Krishna V Shenoy. A central source of movement variability. *Neuron*, 52(6):1085–1096, 2006.

[33] Dominik Straub and Constantin A Rothkopf. Putting perception into action with inverse optimal control for continuous psychophysics. *Elife*, 11:e76635, 2022.

[34] Jonathon W Sensinger and Strahinja Dosen. A review of sensory feedback in upper-limb prostheses from the perspective of human motor control. *Frontiers in neuroscience*, 14:345, 2020.

[35] Dan Liu and Emanuel Todorov. Evidence for the flexible sensorimotor strategies predicted by optimal feedback control. *Journal of Neuroscience*, 27(35):9354–9368, 2007.

[36] Jun Izawa, Tushar Rane, Opher Donchin, and Reza Shadmehr. Motor adaptation as a process of reoptimization. *Journal of Neuroscience*, 28(11):2883–2891, 2008.

[37] Tomohiko Takei, Stephen G Lomber, Douglas J Cook, and Stephen H Scott. Transient deactivation of dorsal premotor cortex or parietal area 5 impairs feedback control of the limb in macaques. *Current Biology*, 31(7):1476–1487, 2021.

[38] Maryam M Shanechi, Ziv M Williams, Gregory W Wornell, Rollin C Hu, Marissa Powers, and Emery N Brown. A real-time brain-machine interface combining motor target and trajectory intent using an optimal feedback control design. *PloS one*, 8(4):e59049, 2013.

[39] Jur Van Den Berg, Pieter Abbeel, and Ken Goldberg. Lqg-mp: Optimized path planning for robots with motion uncertainty and imperfect state information. 2011.

[40] David A Winter. *Biomechanics and motor control of human movement*. John wiley & sons, 2009.

[41] Philipp Karg, Simon Stoll, Simon Rothfuß, and Sören Hohmann. Inverse stochastic optimal control for linear-quadratic gaussian and linear-quadratic sensorimotor control models. In *2022 IEEE 61st Conference on Decision and Control (CDC)*, pages 2801–2808. IEEE, 2022.

[42] Joseph Y Nashed, Frédéric Crevecoeur, and Stephen H Scott. Rapid online selection between multiple motor plans. *Journal of Neuroscience*, 34(5):1769–1780, 2014.

[43] Johannes Friedrich, Siavash Golkar, Shiva Farashahi, Alexander Genkin, Anirvan Sengupta, and Dmitri Chklovskii. Neural optimal feedback control with local learning rules. *Advances in Neural Information Processing Systems*, 34:16358–16370, 2021.

[44] Kaare Brandt Petersen, Michael Syskind Pedersen, et al. The matrix cookbook. *Technical University of Denmark*, 7(15):510, 2008.

[45] Philip Becker-Ehmck, Jan Peters, and Patrick Van Der Smagt. Switching linear dynamics for variational bayes filtering. In *International conference on machine learning*, pages 553–562. PMLR, 2019.

[46] Steven L Brunton, Bingni W Brunton, Joshua L Proctor, and J Nathan Kutz. Koopman invariant subspaces and finite linear representations of nonlinear dynamical systems for control. *PloS one*, 11(2):e0150171, 2016.

# A  Appendix

## A.1  Background: Classic LQAG Solutions

The optimal control and filter gains, $L_{1,\cdots,T-1}$ and $K_{1,\cdots,T-2}$, for the classic LQAG problem — defined by Eqs. 1-2 - can be derived analytically and are given by [8, 1]

$$L_t = -(R_t + B^\mathsf{T} S_{t+1} B)^{-1} B^\mathsf{T} S_{t+1} A \tag{21}$$

$$S_t = Q_t + A^\mathsf{T} S_{t+1}(A - B L_t) \tag{22}$$

$$K_t = A\Sigma_t^e H^\mathsf{T}(H\Sigma_t^e H^\mathsf{T} + \Omega_\omega)^{-1} \tag{23}$$

$$\Sigma_{t+1}^e = \Omega_\xi + (A - K_t H)\Sigma_t^e A^\mathsf{T} . \tag{24}$$

## A.2  Background: Solutions from [1]

The algorithm proposed in [1] alternates between optimizing control and estimation. As for the classic LQAG problem, the control is optimized iteratively in a backward-in-time fashion, while keeping the filters $K_t$ fixed. The solution is derived by using the method of dynamic programming, writing down the Bellman equation for the optimal cost-to-go assuming the unbiasedness of the estimator [1]. From there, the optimal filters $K_t$ are found at fixed $L_t$, again by minimizing the cost-to-go. Taken together, these two optimization steps lead to an iterative algorithm that is supposed to provide the optimal solution to the control problem [1]. For completeness, we present here the optimal solutions for $L_t$ and $K_t$ from [1].

The optimal control gains are given by the following backward algorithm

$$L_t = (R_t + B^\mathsf{T} S_{t+1}^x B + \sum_i C_i^\mathsf{T}(S_{t+1}^x + S_{t+1}^e)C_i)^{-1} B^\mathsf{T} S_{t+1}^x A \tag{25}$$

$$S_t^x = Q_t + A^\mathsf{T} S_{t+1}^x(A - B L_t) + \sum_i D_i^\mathsf{T} K_t^\mathsf{T} S_{t+1}^e K_t D_i \tag{26}$$

$$S_t^e = A^\mathsf{T} S_{t+1}^x B L_t + (A - K_t H)^\mathsf{T} S_{t+1}^e(A - K_t H) \tag{27}$$

with $S_T = Q_T$ and $S_T^e = 0$. Note that in [1] the optimal control law is defined as $u_t = -L_t \hat{x}_t$, whereas we use Eq. 8: to compare the solutions from [1] with ours, we need to invert the sign of the control gains $L_t$.

The optimal filter gains follow instead a forward optimization

$$K_t = A\Sigma_t^e H^\mathsf{T}(H\Sigma_t^e H^\mathsf{T} + \Omega_\omega + \sum_i D_i(\Sigma_t^e + \Sigma_t^{\hat{x}} + \Sigma_t^{\hat{x}e} + \Sigma_t^{e\hat{x}})D_i^\mathsf{T})^{-1} \tag{28}$$

$$\Sigma_{t+1}^e = \Omega_\xi + \Omega_\eta + (A - K_t H)\Sigma_t^e A^\mathsf{T} + \sum_i C_i L_t \Sigma_t^{\hat{x}} L_t^\mathsf{T} C_i^\mathsf{T} \tag{29}$$

$$\begin{aligned}\Sigma_{t+1}^{\hat{x}} = \Omega_\eta + K_t H \Sigma_t^e A^\mathsf{T} + (A - B L_t)\Sigma_t^{\hat{x}}(A - B L_t)^\mathsf{T} + \\ + (A - B L_t)\Sigma_t^{\hat{x}e} H^\mathsf{T} K_t^\mathsf{T} + K_t H \Sigma_t^{e\hat{x}}(A - B L_t)^\mathsf{T}\end{aligned} \tag{30}$$

$$\Sigma_{t+1}^{\hat{x}e} = (A - B L_t)\Sigma_t^{\hat{x}e}(A - K_t H)^\mathsf{T} - \Omega_\eta \tag{31}$$

$$\Sigma_t^{e\hat{x}} = (\Sigma_t^{\hat{x}e})^\mathsf{T} \tag{32}$$

with $\Sigma_1^e = \Sigma_{x_1}$, $\Sigma_1^{\hat{x}} = \hat{x}_1 \hat{x}_1^\mathsf{T}$ and $\Sigma_1^{\hat{x}e} = 0$. Note that in [1] $\Sigma_t^e := \mathbb{E}[e_t e_t^\mathsf{T}]$, where $e_t := x_t - \hat{x}_t$, $\Sigma_t^{\hat{x}} := \mathbb{E}[\hat{x}_t \hat{x}_t^\mathsf{T}]$ and $\Sigma_t^{\hat{x}e} := \mathbb{E}[\hat{x}_t e_t^\mathsf{T}]$.

## A.3  Unbiasedness and Orthogonality Principle: How Internal Noise Affects Optimality

In Section 2.3, we discussed the invalidity of the unbiasedness condition. Here, we provide the details of the one-dimensional problem used to numerically validate this assertion, presenting the plots in Figs. A.3b, c also for the case where $\sigma_\eta = 0.3$. The system parameters are listed in Table 2 in

Appendix A.5.1, where $\sigma_\eta = \sqrt{\Omega_\eta}$. The trial duration is set to $T = 10$ time steps, and we vary $\sigma_\eta$ across the values $0.0, 0.3, 0.6$. At $t = 8$, we compute $\mathbb{E}[x_t|\hat{x}_t]$ as a function of $\hat{x}_t$. To do this, we collect the values of $x_8$ and $\hat{x}_8$ over $5 \cdot 10^7$ trials, bin the data for $\hat{x}_t$ with a bin size of $\delta\hat{x} = 0.1$, and compute the mean of $x_t$ within each bin. The standard deviation is shown as error bars. Note that the choice of $t$ is arbitrary.

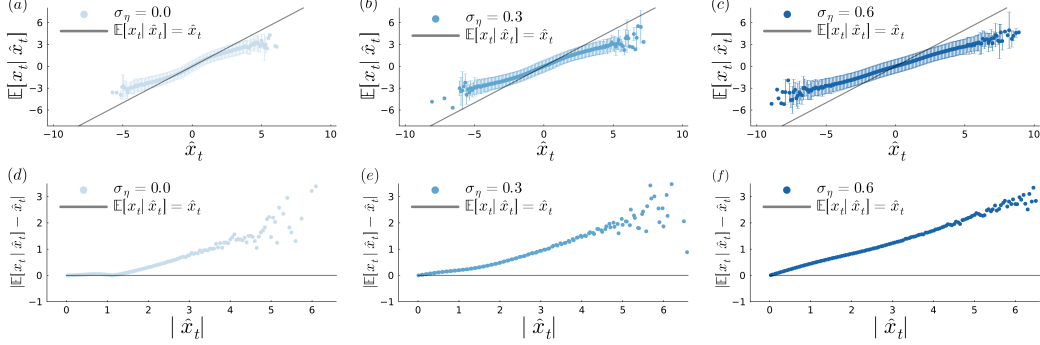

Figure 4: *The invalidity of the unbiasedness condition.* Here we plot $\mathbb{E}[x_t|\hat{x}_t]$, for a given value of $t$, as a function of $\hat{x}_t$ for different $\sigma_\eta$, using the solutions from [1]. The conditional expectation $\mathbb{E}[x_t|\hat{x}_t]$ is computed through Monte Carlo (MC) simulations. **(a)** $\mathbb{E}[x_t|\hat{x}_t]$ as a function of $\hat{x}_t$ for $\sigma_\eta = 0$ (dots with error bars given by the std of our MC estimate). The gray dotted line stands for the bisector, where $\mathbb{E}[x_t|\hat{x}_t] = \hat{x}_t$. **(b)** Same as $(a)$, but for $\sigma_\eta = 0.3$. **(c)** Same as $(a)$ and $(b)$, but for $\sigma_\eta = 0.6$. **(d-f)** Absolute value of the distance between $\mathbb{E}[x_t|\hat{x}_t]$ and $\hat{x}_t$ as a function of $|\hat{x}_t|$ for $\sigma_\eta = 0.0, 0.3, 0.6$. The gray dotted lines represent $\mathbb{E}[x_t|\hat{x}_t] = \hat{x}_t$.

### A.3.1 Orthogonality Principle and Suboptimality

In Section 2.3, we outlined the relationship between the unbiasedness condition and the orthogonality principle, highlighting how this connection results in the suboptimality of the approach in [1] when internal noise is present. For simplicity, and without loss of generality, we consider a 1D scenario, $m = p = k = 1$, where the orthogonality principle implies [8]

$$\Omega_t \equiv \mathbb{E}[\hat{x}_t(x_t - \hat{x}_t)] = \mathbb{E}[\hat{x}_t^2] - \mathbb{E}[x_t\hat{x}_t] = 0 \,. \tag{33}$$

This results in the estimation error being orthogonal to the estimate itself $\hat{x}_t$. We also set $c = d = 1$. We demonstrate here that the condition $\mathbb{E}[x_t|\hat{x}_t] = \hat{x}_t$, used in [1] to derive the optimal control law, implies the orthogonality principle and that this principle cannot be satisfied by an optimal estimator when internal noise is present.

Assuming

$$\mathbb{E}[x_t|\hat{x}_t] = \hat{x}_t \tag{34}$$

and multiplying by $\hat{x}_t$ on both sides and then taking the expectation over $\hat{x}_t$ we obtain

$$\mathbb{E}[\hat{x}_t^2] = \mathbb{E}[x_t\hat{x}_t] \tag{35}$$

corresponding to Eq. 33. In the absence of internal noise, the optimal filter gains $K_t$ can be found by imposing the orthogonality principle, without the need to minimize the cost function. It holds

$$\begin{aligned}
\Omega_{t+1} = {}& (K_t^2 H^2 + K_t^2 D^2 - AK_t H)\mathbb{E}[x_t^2] + \\
& + (A^2 + K_t^2 H^2 + ABL_t - 2AK_t H - BL_t K_t H)\Omega_t + \\
& + (AK_t H - K_t^2 H^2)\mathbb{E}[x_t\hat{x}_t] + K_t^2 \Omega_\omega + \Omega_\eta
\end{aligned} \tag{36}$$

If we use $\Omega_1 = 0$, as in [1] due to the initial conditions, we can solve the equation $\Omega_t = 0$, $\forall t = 1, .., T$, obtaining an equation for $K_t$,

$$K_t = \frac{AH\Gamma_t \pm \sqrt{A^2 H^2 \Gamma_t^2 - 4(H^2\Gamma_t + D^2\mathbb{E}[x_t^2] + \Omega_\omega)\Omega_\eta}}{2(H^2\Gamma_t + D^2\mathbb{E}[x_t^2] + \Omega_\omega)} \tag{37}$$

with

$$\Gamma_t = \mathbb{E}[x_t^2] - \mathbb{E}[x_t \hat{x}_t]. \tag{38}$$

For $\Omega_\eta = 0$, Eq. 37 simplifies to

$$K_t = \frac{AH\Gamma_t \pm AH\Gamma_t}{2(H^2\Gamma_t + D^2\mathbb{E}[x_t^2] + \Omega_\omega)}. \tag{39}$$

Observing that the solution $K_t = 0$ would correspond to an open-loop strategy, sub-optimal (sensory information would not be integrated) for a stochastic partially observable system as the one we are considering, we get for the optimal filter gains

$$K_t^* = \frac{AH\Gamma_t}{H^2\Gamma_t + D^2\mathbb{E}[x_t^2] + \Omega_\omega}. \tag{40}$$

It can be shown that the solution in [1] for $\Omega_\eta = 0$ aligns with Eq. 40. We observe that Eq. 40 can replace Eq. 20 in Algorithm 2 to optimize the filter gains. For $\Omega_\eta = 0$, this leads to the optimal solution. Thus, in the absence of internal noise, the optimization of control and estimation can be performed using two separate objective functions: one enforcing the orthogonality principle for the optimal estimator, and the other minimizing the cost function for the optimal controller, regardless of the multiplicative nature of the noise. This could also be relevant for more biologically plausible scenarios [5].

However, when $\Omega_\eta > 0$, the existence of a real solution for Eq. 37 depends on the initial conditions and is no longer guaranteed. Moreover, as we demonstrate in Appendix A.5.2, the optimal solutions do not satisfy $\Omega_t = 0$ for $\Omega_\eta > 0$. Therefore, the orthogonality principle holds for an optimal Kalman filter only when $\Omega_\eta = 0$. Consequently, the algorithm in [1] assumes unbiased estimation, which should imply the orthogonality principle, even in cases where it no longer applies to the optimal estimator.

**Separation Principle, Orthogonality Principle, Unbiasedness: A Brief Digression**    Unbiasedness, orthogonality, and the separation principle are related but distinct concepts. Here, we briefly clarify their differences and commonalities.

The separation principle stems from the formulation of the classic LQAG problem, where the optimal solutions for control and estimation are mathematically independent, allowing for their separate optimization. However, with multiplicative noise, this independence is lost [1]. We have shown that this breakdown occurs even with additive internal noise and zero multiplicative noises.

The orthogonality principle (in 1D) states that $\mathbb{E}[x_t \hat{x}_t] = \mathbb{E}[\hat{x}_t^2]$, meaning that estimation error and estimate are orthogonal. This condition holds for an optimal Kalman filter only in the absence of internal noise (see Fig. 5a). Internal fluctuations, however, disrupt the mathematical independence between control and estimation, invalidating the orthogonality principle as well. These two concepts are distinct: for instance, with no internal noise but non-zero multiplicative noise, the orthogonality principle would still hold, yet the mathematical independence between control and estimation would be broken.

The unbiasedness condition (which, as previously discussed, never holds) states that $\mathbb{E}[x_t | \hat{x}_t] = \hat{x}_t$, implying the orthogonality principle. This explains the optimality of the solutions in [1] in the absence of internal noise—not due to the validity of the unbiasedness condition, but because the orthogonality condition holds.

## A.4   A Novel Algorithm for Optimal Control Problems

### A.4.1   Derivation of Closed-Form Equations for Moments Propagation

We explicitly derive Eqs. 14-15 here. Notably, no approximations are required to propagate the first two moments of the joint variable $(x, \hat{x})$ in closed form, as both control and estimation are linear in the state and state estimate (see Eqs. 5-8). Consequently, Eqs. 14-15 hold regardless of the

distribution of $(x, \hat{x})$. By taking the expected value of Eqs. 5-7 over the joint distribution of state, state estimate and sensory feedback, we obtain

$$\mathbb{E}[x_{t+1}] = A\mathbb{E}[x_t] + BL_t\mathbb{E}[\hat{x}_t] \tag{41}$$

$$\mathbb{E}[\hat{x}_{t+1}] = K_t H\mathbb{E}[x_t] + (A + BL_t - K_t H)\mathbb{E}[\hat{x}_t] \,, \tag{42}$$

which correspond to Eq. 14. Similarly, we compute the second non-central moments of the joint variable $(x, \hat{x})$, resulting in

$$
\begin{aligned}
\mathbb{E}[x_{t+1}x_{t+1}^\mathsf{T}] &= A\mathbb{E}[x_t x_t^\mathsf{T}]A^\mathsf{T} + BL_t\mathbb{E}[\hat{x}_t \hat{x}_t^\mathsf{T}]L_t^\mathsf{T}B^\mathsf{T} + \\
&\quad + A\mathbb{E}[x_t \hat{x}_t^\mathsf{T}]L_t^\mathsf{T}B^\mathsf{T} + BL_t\mathbb{E}[\hat{x}_t x_t^\mathsf{T}]A^\mathsf{T} + CL_t\mathbb{E}[\hat{x}_t \hat{x}_t^\mathsf{T}]L_t^\mathsf{T}C^\mathsf{T} + \Omega_\xi
\end{aligned} \tag{43}
$$

$$
\begin{aligned}
\mathbb{E}[\hat{x}_{t+1}\hat{x}_{t+1}^\mathsf{T}] &= K_t H\mathbb{E}[x_t x_t^\mathsf{T}]H^\mathsf{T}K_t^\mathsf{T} + (A + BL_t - K_t H)\mathbb{E}[\hat{x}_t \hat{x}_t^\mathsf{T}](A + BL_t - K_t H)^\mathsf{T} + \\
&\quad + K_t H\mathbb{E}[x_t \hat{x}_t^\mathsf{T}](A + BL_t - K_t H)^\mathsf{T} + (A + BL_t - K_t H)\mathbb{E}[\hat{x}_t x_t^\mathsf{T}]H^\mathsf{T}K_t^\mathsf{T} + \\
&\quad + K_t D\mathbb{E}[x_t x_t^\mathsf{T}]D^\mathsf{T}K_t^\mathsf{T} + K_t\Omega_\omega K_t^\mathsf{T} + \Omega_\eta
\end{aligned} \tag{44}
$$

$$
\begin{aligned}
\mathbb{E}[\hat{x}_{t+1}x_{t+1}^\mathsf{T}] &= K_t H\mathbb{E}[x_t x_t^\mathsf{T}]A^\mathsf{T} + (A + BL_t - K_t H)\mathbb{E}[\hat{x}_t \hat{x}_t^\mathsf{T}]L_t^\mathsf{T}B^\mathsf{T} + \\
&\quad + K_t H\mathbb{E}[x_t \hat{x}_t^\mathsf{T}]L_t^\mathsf{T}B^\mathsf{T} + (A + BL_t - K_t H)\mathbb{E}[\hat{x}_t x_t^\mathsf{T}]A^\mathsf{T}
\end{aligned} \tag{45}
$$

$$\mathbb{E}[x_{t+1}\hat{x}_{t+1}^\mathsf{T}] = \mathbb{E}[\hat{x}_{t+1}x_{t+1}^\mathsf{T}]^\mathsf{T} \,. \tag{46}$$

From this, we can derive Eq. 15.

Since the cost function is quadratic in the state and state estimate, the variables $\mu_t$ and $\Sigma_t$, defined in Eqs. 10-11, serve as sufficient statistics to compute $\mathbb{E}[J]$ (Eq. 9), which is all that is needed to derive the optimal control and filter gains.

### A.4.2 Pseudo-Code

For the GD algorithm (Section 3.1) we minimize the expected accumulated cost $\mathbb{E}[J]$, computed through Algorithm 1, with respect to the filter and control gains $L_{1,\cdots,T-1}$, and $K_{1,\cdots,T-2}$, using the function "GradientDescent()" in the "Optim.jl" Julia package. The hyper-parameters of the used algorithms are listed in Table 1 in Appendix A.5.

---

**Algorithm 1** Propagation of the expected cost - GD algorithm

---

1: **Input:** $\mu_1$, $\Sigma_1$ (initial conditions of the system), $L_{1,\cdots,T-1}$, $K_{1,\cdots,T-2}$, and the system parameters $(A, B, H, C_{i=1,\ldots,c}, D_{i=1,\ldots,d}, \Omega_\xi, \Omega_\omega, \Omega_\eta)$.
2: **Output:** $\mathbb{E}[J]$
3: Algorithm steps:
4: $\mathbb{E}[J] = 0$
5: $\mu_{old} = \mu_1$
6: $\Sigma_{old} = \Sigma_1$
7: **for** each iteration $t = 1, 2, \ldots, T$ **do**
8: $\quad E[J] \leftarrow \mathbb{E}[J] + \mathbb{E}[j_t]$, (Eq. 9)
9: $\quad$ Update $M_t$ and $G_t$ (Eqs. 12-13)
10: $\quad \Sigma_{new} = M_t\Sigma_{old}M_t^\mathsf{T} + G_t$
11: $\quad \mu_{new} = M_t\mu_{old}$
12: $\quad \Sigma_{old} \leftarrow \Sigma_{new}$
13: $\quad \mu_{old} \leftarrow \mu_{new}$
14: **end for**

---

### A.5 Experiments: GD Algorithm

The hyper-parameters of all the used algorithms are provided in Table 1. For the GD algorithm (Section 3.1) we minimize the expected accumulated cost $\mathbb{E}[J]$, computed through Algorithm 1, using the function "GradientDescent()" in the "Optim.jl" Julia package.

Table 1: Hyper-parameters of the used algorithms

| Algorithm | Description | value |
|---|---|---|
| GD | Number of iterations of the "GradientDescent()" function | 100000 |
| FPOMP | Number of iterations of the control-estimation optimization | 1000 |
| TOD | Number of iterations of the control-estimation optimization | 1000 |

### A.5.1 One-Dimensional Case: Parameters

We set $c = d = 1$.

Table 2: Parameters of the one-dimensional problem

| Name | Description | value |
|---|---|---|
| $A$ | Linear map for the system dynamics | 1.0 |
| $B$ | Scaling of the control signal | 1.0 |
| $C$ | Scaling matrix for control-dependent noise | 0.5 |
| $D$ | Scaling for signal-dependent noise in the sensory feedback | 0.5 |
| $H$ | Observation matrix | 1 |
| $R_t$ | Control-dependent cost at each $t < T$ | 1 |
| $Q_t$ | Task-related cost at each time $t < T$ | 1 |
| $Q_T$ | Task-related cost at time $t = T$ | 20 |
| $T$ | time steps | 100 |
| $\mathbb{E}[\hat{x}_1] = \mathbb{E}[x_1]$ | Initial condition for the mean state and state estimate | 1.0 |
| $\Sigma_{x_1}$ | Initial covariance of the state | 0.0 |
| $\Sigma_{\hat{x}_1}$ | Initial covariance of the state estimate | 0.0 |
| $\Omega_\xi$ | Covariance matrix of the additive Gaussian noise $\xi_t$ | $0.5^2$ |
| $\Omega_\omega$ | Covariance matrix of the additive Gaussian noise $\omega_t$ | $0.5^2$ |
| $\sigma_\eta$ | Standard deviation of the additive internal Gaussian noise $\eta_t$ | $\{0.0 : 0.1 : 2.0\}$ |

### A.5.2 One-Dimensional Case: Understanding the Qualitative Differences

**When the Orthogonality Principle Is No Longer Optimal** As discussed in Section 2.3 and Appendix A.3.1, the presence of internal noise causes the optimal estimator to no longer satisfy the orthogonality principle. Here, we demonstrate this result numerically, using the same one-dimensional problem presented in Section 3.2.

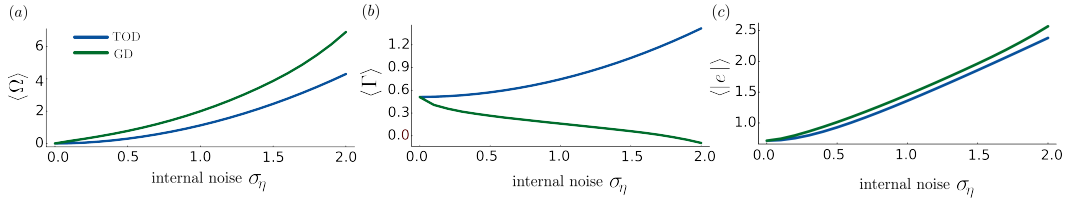

Figure 5: *Filtering out the internal fluctuations.* **(a)** $\Omega_t$, averaged over time (we indicate the time average with $\langle \cdot \rangle$), as a function of $\sigma_\eta$ for TOD and GD algorithms. **(b)** $\langle \Gamma \rangle$ as a function of $\sigma_\eta$. **(c)** $\langle e \rangle$ as a function of $\sigma_\eta$. The error bars (mean $\pm$ 1SEM from Monte Carlo simulations) are not visible as too small.

The optimal solutions do not minimize $\Omega_t = \mathbb{E}[\hat{x}_t^2] - \mathbb{E}[x_t \hat{x}_t]$ (Fig. 5a). Instead, the optimal strategy appears to favor lower values of $\Gamma_t = \mathbb{E}[x_t^2] - \mathbb{E}[x_t \hat{x}_t]$ (Fig. 5b). This allows the system to filter out internal fluctuations affecting the estimation process, reducing their correlation with the latent state dynamics, $x$. As a result, the absolute estimation error, $|e_t| = \sqrt{\mathbb{E}[(x_t - \hat{x}_t)^2]} = \sqrt{\Omega_t + \Gamma_t}$, is slightly (but significantly) larger for the GD solutions (Fig. 5c), which seems to help decorrelate internal noise from the state evolution.

This 'decorrelation mechanism' is achieved through an intertwined modulation of control and filter gains. In the next paragraph, we provide a geometric interpretation of this behavior through an eigenvector decomposition of the dynamical system under investigation.

**Eigenvector Decomposition and Adaptability of the Solutions** In one dimension we can write the update equations for

$$\Gamma_t = \mathbb{E}[x_t^2] - \mathbb{E}[x_t \hat{x}_t] \tag{47}$$

$$\Omega_t = \mathbb{E}[\hat{x}_t^2] - \mathbb{E}[x_t \hat{x}_t] \tag{48}$$

as

$$\begin{pmatrix} \Gamma_{t+1} \\ \Omega_{t+1} \end{pmatrix} = \mathcal{M}_t \begin{pmatrix} \Gamma_t \\ \Omega_t \end{pmatrix} + \begin{pmatrix} \Omega_\xi + C^2 L_t^2 \mathbb{E}[\hat{x}_t^2] \\ \Omega_\eta + K_t^2 \Omega_\omega + K_t^2 D^2 \mathbb{E}[x_t^2] \end{pmatrix} \tag{49}$$

where

$$\mathcal{M}_t = (A - K_t H) \begin{pmatrix} A & -BL_t \\ -K_t H & A + BL_t - K_t H \end{pmatrix}. \tag{50}$$

The eigenvectors of $\mathcal{M}_t$ are given by

$$\vec{w}_1 = \begin{pmatrix} -1 \\ 1 \end{pmatrix} \tag{51}$$

$$\vec{w}_2 = \begin{pmatrix} BL_t/K_t H \\ 1 \end{pmatrix}. \tag{52}$$

Note that the angles $\theta_t$ between these two eigenvectors are the same as the angles between the eigenvectors of the matrix $M_t$. Indeed, the eigenvectors of $M_t$ are given by

$$\vec{v}_1 = \begin{pmatrix} 1 \\ 1 \end{pmatrix} \tag{53}$$

$$\vec{v}_2 = \begin{pmatrix} -BL_t/K_t H \\ 1 \end{pmatrix}. \tag{54}$$

A parity operation (along the $x$-axis) maps ones into the others, preserving the angles.

The optimal solution arises from the adjustment of the angle $\theta$ between the two eigenvectors (in this one-dimensional case). As $\sigma_\eta$ increases, $\theta$ also increases, due to the joint modulation of $L_t$ and $K_t$ with $\sigma_\eta$ (Fig. 6a). This increase in $\theta$ allows the system to filter internal fluctuations more effectively and better generalize to other levels of $\sigma_\eta$ (Fig. 6b).

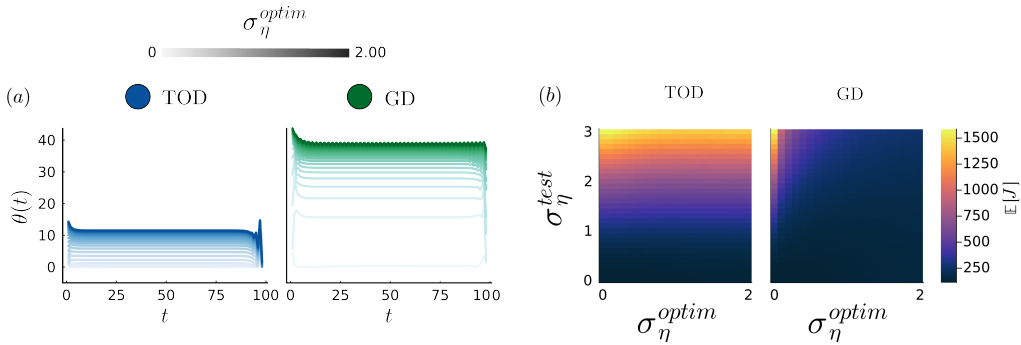

Figure 6: *Intertwined modulation of control and filter gains to deal with the internal noise.* **(a)** Angles $\theta_t$ between the two eigenvectors of the matrix $M_t$ (Eq. 12 and see next Paragraph), at different levels of internal noise $\sigma_\eta^{optim}$, for TOD and GD algorithms. **(b)** "Adaptability" of the two solutions; the solution found by the GD algorithm (right panel) generalizes better than the one by TOD (left panel) when optimized for a certain level of internal noise, $\sigma_\eta^{optim}$, and tested on another one, $\sigma_\eta^{test}$: for larger $\sigma_\eta^{optim}$, the generalization property improves thanks to due modulation of $\theta_t$.

By examining the modulation of $\vec{w}_2$ as $\sigma_\eta$ changes in the $(\Gamma_t - \Omega_t)$ plane, we can offer a heuristic interpretation of the different solutions found by the TOD and GD algorithms. As $\sigma_\eta$ increases,

the angle between $\vec{w}_1$ and $\vec{w}_2$ grows for both algorithms. However, this modulation is much more pronounced in the GD solution (Fig. 6a).

Furthermore, if only additive noise were considered, there would be no modulation of $\theta_t$ with $\sigma_\eta$ in the TOD solution (for confirmation, see Appendix A.8.2: without multiplicative noise, TOD's derivation does not modulate the control gains with $\sigma_\eta$).

The joint modulation of $L_t$ and $K_t$ causes $\vec{w}_2$ to move closer to the $y$-axis in the GD solution (Fig. 7, green line). This configuration results in more effective filtering of internal fluctuations, decoupling them from the latent state dynamics, since these fluctuations occur on $\Omega_t$ (see Eq. 49). This result aligns with the observed decrease of $L_t$ as $\sigma_\eta$ increases (see Figs. 2 and 3), where lowering the control gain moves $\vec{w}_2$ closer to the $y$-axis. Thus, this eigenvector analysis qualitatively explains the trends observed in Fig. 5 for $\langle \Gamma \rangle$ as a function of $\sigma_\eta$ in both the TOD and GD solutions.

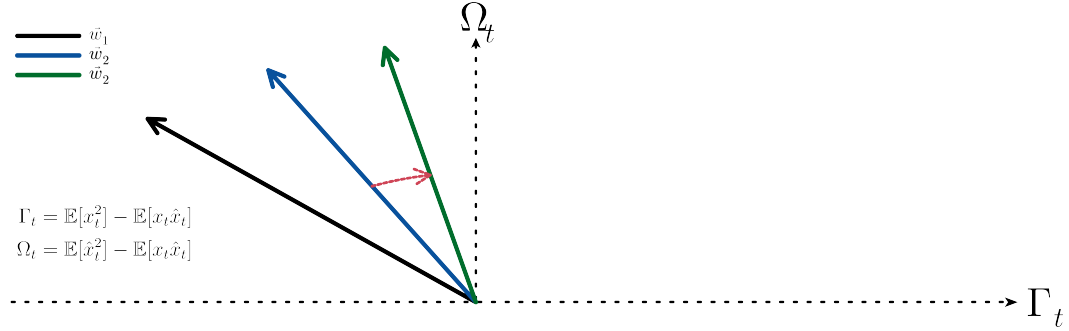

Figure 7: *Eigenvector decomposition of the dynamics*. We show here a qualitative representation of the eigenvectors of the matrix $\mathcal{M}_t$ in the plane $(\Gamma_t, \Omega_t)$. The black arrow represents the "shared" eigenvector $\vec{w}_1$, while the blue (green) arrow represents $\vec{w}_2$ for TOD (GD) solution. Note that the optimal $L_t$ are negative, while the optimal $K_t$ are positive (Fig. 2).

### A.5.3 One-Dimensional Case: Improving Performance Without Internal Noise

We briefly show that, even in the absence of internal noise, if the algorithm has not yet converged, the solution proposed by [1] does not yield the optimal control law. We demonstrate this in a one-dimensional example, using the same parameters shown in Appendix A.5.1 (but the result is valid in general), while only varying the scaling matrix for the multiplicative sensory noise $D$ and keeping $\sigma_\eta = 0$. We fix the filter gains at the suboptimal constant value $K_{1,\cdots,T-2} = A = 1.0$, and optimize the vector $L_{1,\cdots,T-1}$ using TOD and GD algorithms.

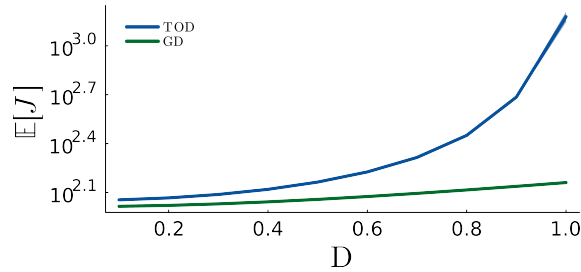

Figure 8: *Enhanced performance when optimizing control at fixed filter gains and zero internal noise*. We plot the expected accumulated cost $\mathbb{E}[J]$, computed by averaging the quantity from Eq. 3 over $50k$ trials, as a function of the scaling matrix $D$, with error bars (mean $\pm$ 1SEM from Monte Carlo simulations, error bars not visible as too small), for the two algorithms TOD and GD.

We find that TOD control law leads to a higher expected accumulated cost $\mathbb{E}[J]$ (Fig. 8). The improved performance of the GD (and FPOMP - in Section 3.2 we show that the solutions of the two match) algorithm arises from not assuming unbiasedness when optimizing control. As a result, the

algorithm adjusts the control gains to account for the bias introduced by the suboptimal estimator. Similar to the case with internal noise, the control gains found by the GD algorithms in this scenario are typically smaller than those found by the algorithm of [1]. A similar behavior occurs when optimizing the filter gains while keeping the control gains fixed and suboptimal. The difference in performance is due to the fact that the derivation of the optimal estimator in [1] is only valid when the control gains are optimal. Thus, the algorithm in [1] does not apply when the suboptimality of the controller needs to be 'balanced' by the estimator. When the algorithm in [1] is run in its entirety, optimizing both control and filter gains iteratively, these issues are resolved. Upon convergence, the controller and estimator are optimally adjusted to each other, in the absence of internal noise.

### A.5.4  Multi-Dimensional Case: Parameters

For the sensorimotor task described in Section 3.2, the discrete-time dynamics is the same as in [1],

$$p(t + \Delta t) = p(t) + \dot{p}(t)\Delta t \tag{55}$$
$$\dot{p}(t + \Delta t) = \dot{p}(t) + f(t)\Delta t/m \tag{56}$$
$$f(t + \Delta t) = f(t)(1 - \Delta t/\tau_2) + g(t)\Delta t/\tau_2 \tag{57}$$
$$g(t + \Delta t) = g(t)(1 - \Delta t/\tau_1) + u(t)(1 + \sigma_\varepsilon \varepsilon_t)\Delta t/\tau_1 \tag{58}$$

We have therefore the following system parameters (with $c = d = 1$)

$$A = \begin{pmatrix} 1 & \Delta t & 0 & 0 \\ 0 & 1 & \Delta t/m & 0 \\ 0 & 0 & 1 - \Delta t/\tau_2 & \Delta t/\tau_2 \\ 0 & 0 & 0 & 1 - \Delta t/\tau_1 \end{pmatrix} \tag{59}$$

$$B = \begin{pmatrix} 0 & 0 & 0 & \Delta t/\tau_1 \end{pmatrix}^\mathsf{T} \tag{60}$$

$$C = \begin{pmatrix} 0 & 0 & 0 & \sigma_\varepsilon \Delta t/\tau_1 \end{pmatrix}^\mathsf{T} \tag{61}$$

$$H = \begin{pmatrix} 1 & 0 & 0 & 0 \\ 0 & 0 & 0 & 0 \\ 0 & 0 & 0 & 0 \\ 0 & 0 & 0 & 0 \end{pmatrix} \tag{62}$$

$$D = \begin{pmatrix} \sigma_\rho & 0 & 0 & 0 \\ 0 & 0 & 0 & 0 \\ 0 & 0 & 0 & 0 \\ 0 & 0 & 0 & 0 \end{pmatrix} \tag{63}$$

$$Q_{1,\cdots,T-1} = \begin{pmatrix} 0 & 0 & 0 & 0 \\ 0 & 0 & 0 & 0 \\ 0 & 0 & 0 & 0 \\ 0 & 0 & 0 & 0 \end{pmatrix} \tag{64}$$

$$Q_T = \vec{p}\vec{p}^\mathsf{T} + \vec{v}\vec{v}^\mathsf{T} + \vec{f}\vec{f}^\mathsf{T} \tag{65}$$

$$R_{1,\cdots,T-1} = \frac{r}{T-1} \tag{66}$$

$$R_T = 0 \tag{67}$$

$$\vec{p} = \begin{pmatrix} 1 & 0 & 0 & 0 \end{pmatrix} \tag{68}$$

$$\vec{v} = \begin{pmatrix} 0 & w_v & 0 & 0 \end{pmatrix} \tag{69}$$

$$\vec{f} = \begin{pmatrix} 0 & 0 & w_v & 0 \end{pmatrix} \tag{70}$$

$$\Omega_\xi = \begin{pmatrix} 0 & 0 & 0 & 0 \\ 0 & \sigma_\xi^2 & 0 & 0 \\ 0 & 0 & 0 & 0 \\ 0 & 0 & 0 & 0 \end{pmatrix} \tag{71}$$

$$\Omega_\omega = \sigma_\omega^2 \tag{72}$$

$$\Omega_\eta = \begin{pmatrix} \sigma_\eta^2 & 0 & 0 & 0 \\ 0 & \sigma_{\eta_v}^2 & 0 & 0 \\ 0 & 0 & \sigma_{\eta_f}^2 & 0 \\ 0 & 0 & 0 & \sigma_{\eta_c}^2 \end{pmatrix} \tag{73}$$

with the initial conditions given by

$$\mathbb{E}[x_1] = (z \quad 0 \quad 0 \quad 0)^\mathsf{T} \tag{74}$$

$$\mathbb{E}[\hat{x}_1] = \mathbb{E}[x_1] \tag{75}$$

$$\Sigma_{x_1} = \begin{pmatrix} \sigma_z^2 & 0 & 0 & 0 \\ 0 & 0 & 0 & 0 \\ 0 & 0 & 0 & 0 \\ 0 & 0 & 0 & 0 \end{pmatrix} \tag{76}$$

$$\Sigma_{\hat{x}_1} = \begin{pmatrix} 0 & 0 & 0 & 0 \\ 0 & 0 & 0 & 0 \\ 0 & 0 & 0 & 0 \\ 0 & 0 & 0 & 0 \end{pmatrix}. \tag{77}$$

The parameters of the problem are listed in Table 3 (std = standard deviation).

Table 3: Parameters of the sensorimotor task

| Name | Description | value |
|---|---|---|
| $\Delta t$ | time-step $(s)$ | 0.010 |
| $m$ | mass of the hand $(Kg)$, modelled as a point mass | 1 |
| $\tau_1$ | time constant of the second order low pass filter | 0.04 |
| $\tau_2$ | time constant of the second order low pass filter | 0.04 |
| $r$ | Control-dependent cost at each $t < T$ | $1e^{-5}$ |
| $w_v$ | Task-related cost at time $t = T$ for the velocity | 0.2 |
| $w_f$ | Task-related cost at time $t = T$ for the force | 0.01 |
| $T$ | time steps | 100 |
| $z$ | Target position | 0.15 |
| $\sigma_z$ | Target position standard deviation | 0.0 |
| $\sigma_\xi$ | std of the additive Gaussian noise $\xi_t$ | 0.0 |
| $\sigma_\omega$ | std of the additive Gaussian noise $\omega_t$ | 0.0 |
| $\sigma_\varepsilon$ | std of the control-dependent noise $\varepsilon_t$ | 0.5 |
| $\sigma_\rho$ | std of the signal-dependent noise $\rho$ | 0.5 |
| $\sigma_\eta$ | std of the additive internal noise $\eta_t$ for the position estimate | $\{0.0, 0.005, 0.05, 0.5\}$ |
| $\sigma_{\eta_v}$ | std of the additive internal noise $\eta_t$ acting on the velocity estimate | 0 |
| $\sigma_{\eta_f}$ | std of the additive internal noise $\eta_t$ for the force estimate | 0 |
| $\sigma_{\eta_g}$ | std of the additive internal noise $\eta_t$ for the estimate of $g$ | 0 |

Note that the initial condition for the state $x_1$ is the actual target position: in such a way the control signal $u_t$ aims at minimizing the distance from $x_t = 0$.

### A.5.5  Multi-Dimensional Case: Impact of Internal Noise

We provide here an extended analysis of the impact of internal noise in the sensorimotor task discussed in Section 3.2, offering additional insights. We compute the posterior variance of $x_t$ (state belief variance) to assess how internal noise affects it. For $\sigma_\eta = 0, 0.005, 0.05, 0.5$ at $t = 80$ (an arbitrarily chosen time), we find $\sigma^2[x_t] = 2 \cdot 10^{-5}, 5 \cdot 10^{-5}, 8 \cdot 10^{-5}, 2 \cdot 10^{-4}$, respectively. In the case of $\sigma_\eta = 0.05$, where internal noise accounts for about 10%, there is a significant increase in state uncertainty compared to the scenario without internal noise. Indeed, at $t = 80$, since both the state $x_t$ and the state estimate $\hat{x}_t$ are near zero (with the target position as the reference point in our coordinate system), internal noise becomes the dominant source of fluctuations, as we are only accounting for multiplicative noise in this task.

### A.5.6  Scaling to Higher-Dimensional Problems: An Application

We demonstrate how our algorithm scales to high-dimensional problems, building on the discussion in the final paragraph of Section 3.2. We implement a high-dimensional task to show the generalizability of the GD algorithm. The same results would apply to its analytical counterpart, the FPOMP algorithm, as discussed in Section 3.3, and Appendices A.8.1, A.8.2. In this scenario, we set the

dimensions of the state, control, and observation to $m = 10$, $p = 4$, and $k = 10$, respectively. Note that this significantly increases the dimensionality compared to the problem in Section 3.2 (for the multi-dimensional case).

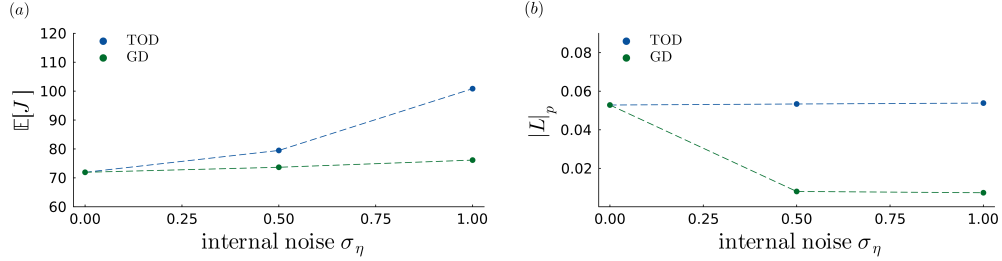

Figure 9: *High-dimensional task.* **(a)** Expected accumulated cost as a function of $\sigma_\eta$ for TOD (blue dots) and GD (green dots) algorithms. We see that even in this high-dimensional task, GD solutions outperform the ones from [1]. To compute the expected cost, we used Algorithm 1 (but the results are confirmed by Monte Carlo simulations). **(b)** Pseudo-determinant of the control gains $L$ (averaged over time), denoted as $|L|_p$ as a function of $\sigma_\eta$ for TOD (blue dots) and GD (green dots) algorithms.

The system matrices $A$, $B$, and $D$ are random matrices with elements drawn from a standard normal distribution (mean zero, standard deviation one), while $C$ is defined as $C = \sigma_\varepsilon B$. The matrix $H$ is the identity matrix, and the time horizon is set to $T = 10$. All elements of the state and state estimate vectors are initialized to one. We used $\sigma_\xi = \sigma_\omega = \sigma_\rho = \sigma_\varepsilon = 0.5$ and varied $\sigma_\eta$ across values of 0.0, 0.5, and 1.0. The matrices defining the quadratic cost functions, $Q$ and $R$, are identity matrices at each time step. All the findings from Section 3.2 are confirmed in this high-dimensional setting (Fig. 9). The GD algorithm continues to outperform the solutions in [1], with performance improving as internal noise increases, and the control gain magnitude decreases as internal fluctuations grow. In fact, as internal noise increases, the optimal strategy involves reducing control over the system. To quantify control magnitude, we compute the pseudo-determinant of $L_{1,\cdots,T-1}$ and average it over time. The pseudo-determinant, a generalization of the determinant for non-square matrices, provides a measure of the volume scaling induced by the control gains.

## A.6 FPOMP Algorithm: An Analytical Counterpart to the Numerical GD Algorithm

### A.6.1 One-Dimensional Case

In the one-dimensional case we have $m = p = k = 1$. Additionally, to simplify the notation, we set $c = d = 1$. We start by defining

$$\vec{F}_t = \begin{pmatrix} F_{t,1} \\ F_{t,2} \\ F_{t,3} \end{pmatrix} = \begin{pmatrix} A^2 \\ (B^2 + C^2)L_t^2 \\ 2ABL_t \end{pmatrix} \tag{78}$$

$$\vec{G}_t = \begin{pmatrix} G_{t,1} \\ G_{t,2} \\ G_{t,3} \end{pmatrix} = \begin{pmatrix} K_t^2(H^2 + D^2) \\ (A + BL_t)^2 + K_t^2 H^2 - 2AK_t H - 2BL_t K_t H \\ 2BL_t K_t H + 2AK_t H - 2K_t^2 H^2 \end{pmatrix} \tag{79}$$

$$\vec{H}_t = \begin{pmatrix} H_{t,1} \\ H_{t,2} \\ H_{t,3} \end{pmatrix} = \begin{pmatrix} AK_t H \\ ABL_t + B^2 L_t^2 - BL_t K_t H \\ A^2 + ABL_t - AK_t H + BL_t K_t H \end{pmatrix}. \tag{80}$$

In one dimension, we can then propagate the non-central moments as

$$\mathbb{E}[x_{t+1}^2] = F_{t,1}\mathbb{E}[x_t^2] + F_{t,2}\mathbb{E}[\hat{x}_t^2] + F_{t,3}\mathbb{E}[x_t\hat{x}_t] + \Omega_\xi \tag{81}$$

$$\mathbb{E}[\hat{x}_{t+1}^2] = G_{t,1}\mathbb{E}[x_t^2] + G_{t,2}\mathbb{E}[\hat{x}_t^2] + G_{t,3}\mathbb{E}[x_t\hat{x}_t] + K_t^2\Omega_\omega + \Omega_\eta \tag{82}$$

$$\mathbb{E}[x_{t+1}\hat{x}_{t+1}] = H_{t,1}\mathbb{E}[x_t^2] + H_{t,2}\mathbb{E}[\hat{x}_t^2] + H_{t,3}\mathbb{E}[x_t\hat{x}_t]. \tag{83}$$

The derivatives of the non-central moments with respect to $L_t$ and $K_t$ obey the following equations, for $i = 1, ..., T$,

$$\frac{\partial \mathbb{E}[x_{t+i}^2]}{\partial L_t} = a_{t+i-1,1}L_t + b_{t+i-1,1} \tag{84}$$

$$\frac{\partial \mathbb{E}[\hat{x}_{t+i}^2]}{\partial L_t} = a_{t+i-1,2}L_t + b_{t+i-1,2} \tag{85}$$

$$\frac{\partial \mathbb{E}[x_{t+i}\hat{x}_{t+i}]}{\partial L_t} = a_{t+i-1,3}L_t + b_{t+i-1,3} \tag{86}$$

and

$$\frac{\partial \mathbb{E}[x_{t+i}^2]}{\partial K_t} = \alpha_{t+i-1,1}K_t + \beta_{t+i-1,1} \tag{87}$$

$$\frac{\partial \mathbb{E}[\hat{x}_{t+i}^2]}{\partial K_t} = \alpha_{t+i-1,2}K_t + \beta_{t+i-1,2} \tag{88}$$

$$\frac{\partial \mathbb{E}[x_{t+i}\hat{x}_{t+i}]}{\partial K_t} = \alpha_{t+i-1,3}K_t + \beta_{t+i-1,3} , \tag{89}$$

with $\vec{a}$, $\vec{b}$, $\vec{\alpha}$ and $\vec{\beta}$ given by the following recursive equations

$$\vec{a}_{t+1} = \begin{pmatrix} a_{t+1,1} \\ a_{t+1,2} \\ a_{t+1,3} \end{pmatrix} = \begin{pmatrix} \vec{F}_{t+1} \cdot \vec{a}_t \\ \vec{G}_{t+1} \cdot \vec{a}_t \\ \vec{H}_{t+1} \cdot \vec{a}_t \end{pmatrix} \tag{90}$$

$$\vec{b}_{t+1} = \begin{pmatrix} b_{t+1,1} \\ b_{t+1,2} \\ b_{t+1,3} \end{pmatrix} = \begin{pmatrix} \vec{F}_{t+1} \cdot \vec{b}_t \\ \vec{G}_{t+1} \cdot \vec{b}_t \\ \vec{H}_{t+1} \cdot \vec{b}_t \end{pmatrix} \tag{91}$$

$$\vec{\alpha}_{t+1} = \begin{pmatrix} \alpha_{t+1,1} \\ \alpha_{t+1,2} \\ \alpha_{t+1,3} \end{pmatrix} = \begin{pmatrix} \vec{F}_{t+1} \cdot \vec{\alpha}_t \\ \vec{G}_{t+1} \cdot \vec{\alpha}_t \\ \vec{H}_{t+1} \cdot \vec{\alpha}_t \end{pmatrix} \tag{92}$$

$$\vec{\beta}_{t+1} = \begin{pmatrix} \beta_{t+1,1} \\ \beta_{t+1,2} \\ \beta_{t+1,3} \end{pmatrix} = \begin{pmatrix} \vec{F}_{t+1} \cdot \vec{\beta}_t \\ \vec{G}_{t+1} \cdot \vec{\beta}_t \\ \vec{H}_{t+1} \cdot \vec{\beta}_t \end{pmatrix} . \tag{93}$$

The initial conditions for Eqs. 90-93 are

$$\vec{a}_t = \begin{pmatrix} 2(B^2 + C^2)\mathbb{E}[\hat{x}_t^2] \\ 2B^2\mathbb{E}[\hat{x}_t^2] \\ 2B^2\mathbb{E}[\hat{x}_t^2] \end{pmatrix} \tag{94}$$

$$\vec{b}_t = \begin{pmatrix} 2AB\mathbb{E}[x_t\hat{x}_t] \\ 2AB\mathbb{E}[\hat{x}_t^2] - 2BK_tH(\mathbb{E}[\hat{x}_t^2] - \mathbb{E}[x_t\hat{x}_t]) \\ AB(\mathbb{E}[\hat{x}_t^2] + \mathbb{E}[x_t\hat{x}_t]) - BK_tH(\mathbb{E}[\hat{x}_t^2] - \mathbb{E}[x_t\hat{x}_t]) \end{pmatrix} \tag{95}$$

$$\vec{\alpha}_t = \begin{pmatrix} 0 \\ 2H^2(\mathbb{E}[x_t^2] + \mathbb{E}[\hat{x}_t^2] - 2\mathbb{E}[x_t\hat{x}_t]) + 2\Omega_\omega + 2D^2\mathbb{E}[x_t^2] \\ 0 \end{pmatrix} \tag{96}$$

$$\vec{\beta}_t = \begin{pmatrix} 0 \\ -2H(A + BL_t)(\mathbb{E}[\hat{x}_t^2] - \mathbb{E}[x_t\hat{x}_t]) \\ AH(\mathbb{E}[x_t^2] - \mathbb{E}[x_t\hat{x}_t]) - BL_tH(\mathbb{E}[\hat{x}_t^2] - \mathbb{E}[x_t\hat{x}_t]) \end{pmatrix} . \tag{97}$$

By observing that the expected accumulated cost, Eq. 9 (adapted to the one-dimensional case), will be a function of $\mathbb{E}[x_t^2]$ and $\mathbb{E}[\hat{x}_t^2]$, for $t = 1, ..., T - t$, and by using Eqs. 84-89, we can rewrite Eqs.

17-18 as

$$\frac{\partial}{\partial L_t} \sum_{i=0}^{T-t} \mathbb{E}[j_{t+i}|\mu_t, \Sigma_t] = 2R_t\mathbb{E}[\hat{x}_t^2]L_t+$$

$$+ \sum_{i=1}^{T-t}[(Q_{t+i}a_{t+i-1,1} + R_{t+i}L_{t+i}^2 a_{t+i-1,2})L_t+ \tag{98}$$

$$+ (Q_{t+i}b_{t+i-1,1} + R_{t+i}L_{t+i}^2 b_{t+i-1,2})] = 0$$

and

$$\frac{\partial}{\partial K_t} \sum_{i=0}^{T-t} \mathbb{E}[j_{t+i}|\mu_t, \Sigma_t] = \sum_{i=1}^{T-t}[(Q_{t+i}\alpha_{t+i-1,1} + R_{t+i}L_{t+i}^2 \alpha_{t+i-1,2})K_t+$$

$$+ (Q_{t+i}\beta_{t+i-1,1} + R_{t+i}L_{t+i}^2 \beta_{t+i-1,2})] = 0 \ . \tag{99}$$

Therefore, from Eqs. 98-99, we have the following instantiations of Eqs. 19-20 for the optimal control and filter gains at time $t$, $L_t^*$ and $K_t^*$,

$$L_t^* = -\frac{L_t^{num}}{L_t^{den}} \tag{100}$$

$$K_t^* = -\frac{K_t^{num}}{K_t^{den}} \ , \tag{101}$$

with

$$L_t^{num} = \sum_{i=1}^{T-t} \left( Q_{t+i}b_{t+i-1,1} + R_{t+i}L_{t+i}^2 b_{t+i-1,2} \right) , \tag{102}$$

$$L_t^{den} = 2R_t\mathbb{E}[\hat{x}_t^2]+$$

$$+ \sum_{i=1}^{T-t}(Q_{t+i}a_{t+i-1,1} + R_{t+i}L_{t+i}^2 a_{t+i-1,2}) \tag{103}$$

and

$$K_t^{num} = \sum_{i=1}^{T-t} \left( Q_{t+i}\beta_{t+i-1,1} + R_{t+i}L_{t+i}^2 \beta_{t+i-1,2} \right) , \tag{104}$$

$$K_t^{den} = \sum_{i=1}^{T-t} \left( Q_{t+i}\alpha_{t+i-1,1} + R_{t+i}L_{t+i}^2 \alpha_{t+i-1,2} \right) . \tag{105}$$

We can then use Eqs. 100-101, to implement Algorithm 2 and extract $L_{1,\cdots,T-1}^*$, and $K_{1,\cdots,T-2}^*$, for the one-dimensional problem.

### A.6.2 Multi-Dimensional Case

For the multi-dimensional case, we derive Eqs. 19-20 for the classic LQAG problem ($C_i = 0$ for $i = 1,..,c$ and $D_i = 0$ for $i = 1,..,d$) in the presence of internal noise ($\Omega_\eta \geq 0$).

As a title of example, we derive here Eq. 19 for the optimal $L_t^*$ (to be used in Algorithm 2), but the approach would be the same for the optimal filter gains $K_t^*$. The extension to the more general scenario including the multiplicative sources of noise would follow a similar method. As outlined in Section 3.1, Eq. 9, the expected cost per step is given by

$$\mathbb{E}[j_{t+i}] = \mathbb{E}[x_{t+i}]^\mathsf{T} Q_{t+i}\mathbb{E}[x_{t+i}] + \mathbb{E}[\hat{x}_{t+i}]^\mathsf{T} L_t^\mathsf{T} R_t L_{t+i}\mathbb{E}[\hat{x}_{t+i}]+$$

$$+ Tr[Q_{t+i}\Sigma_{x_{t+i}}] + Tr[L_{t+i}^\mathsf{T} R_{t+i}L_{t+i}\Sigma_{\hat{x}_{t+i}}], \tag{106}$$

for $i = 0, ..., T - t$.

When computing $\mathbb{E}[j_{t+i}|\mu_t, \Sigma_t]$ to write down Eq. 17 (with $C_i = 0$, $i = 1,...,c$ and $D_i = 0$, $i = 1,...,d$), and derive Eq. 19, the coefficients multiplying $\mathbb{E}[\hat{x}_t]\mathbb{E}[\hat{x}_t]^\mathsf{T}$ coming from the term

$\mathbb{E}[x_{t+i}]^{\mathsf{T}}Q_{t+i}\mathbb{E}[x_{t+i}]$ in Eq. 106 will be the same as the ones multiplying $\Sigma_{\hat{x}_t}$ and coming from the term $Tr[Q_{t+i}\Sigma_{x_{t+i}}]$. The same holds for the coefficients multiplying respectively $\mathbb{E}[x_t]\mathbb{E}[\hat{x}_t]^{\mathsf{T}}$ and $\Sigma_{x_t,\hat{x}_t}$.

Similarly, we can group together the coefficients coming from the other two factors $\mathbb{E}[\hat{x}_{t+i}]^{\mathsf{T}}L_t^{\mathsf{T}}R_tL_{t+i}\mathbb{E}[\hat{x}_{t+i}]$ and $Tr[L_{t+i}^{\mathsf{T}}R_{t+i}L_{t+i}^{\mathsf{T}}\Sigma_{\hat{x}_{t+i}}]$ in Eq. 106.

We now note that the terms dependent on $L_t$ appearing in $\mathbb{E}[j_{t+i}|\mu_t, \Sigma_t]$ will show a dependence on the afore-mentioned moments $\mathbb{E}[\hat{x}_t]\mathbb{E}[\hat{x}_t]^{\mathsf{T}}$, $\Sigma_{\hat{x}_t}$, $\mathbb{E}[x_t]\mathbb{E}[\hat{x}_t]^{\mathsf{T}}$ and $\Sigma_{x_t,\hat{x}_t}$. More specifically, the quadratic factors in $L_t$ will only depend on $\mathbb{E}[\hat{x}_t]\mathbb{E}[\hat{x}_t]^{\mathsf{T}}$ and $\Sigma_{\hat{x}_t}$. Taken together, these observations lead to the following form for Eq. 17,

$$\mathcal{J}_t L_t^* \mathbb{E}[\hat{x}_t \hat{x}_t^{\mathsf{T}}] + \mathcal{S}_t \mathbb{E}[x_t \hat{x}_t^{\mathsf{T}}] + \mathcal{P}_t \mathbb{E}[\hat{x}_t \hat{x}_t^{\mathsf{T}}] = 0, \tag{107}$$

where we have used $\Sigma_{\hat{x}_t} + \mathbb{E}[\hat{x}_t]\mathbb{E}[\hat{x}_t]^{\mathsf{T}} = \mathbb{E}[\hat{x}_t \hat{x}_t^{\mathsf{T}}]$ and $\Sigma_{x_t,\hat{x}_t} + \mathbb{E}[x_t]\mathbb{E}[\hat{x}_t]^{\mathsf{T}} = \mathbb{E}[x_t \hat{x}_t^{\mathsf{T}}]$.

Therefore, to find the optimal control gains $L_t^*$ from Eq. 107, we only need to compute the coefficients $\mathcal{J}_t$, $\mathcal{S}_t$ and $\mathcal{P}_t$, similar to what we have done for the one-dimensional case in Appendix A.6.1. As before, we can compute the coefficients $\mathcal{J}_t$, $\mathcal{S}_t$ and $\mathcal{P}_t$ by only looking at the first two terms appearing in Eq. 106, that is $\mathbb{E}[x_{t+i}]^{\mathsf{T}}Q_{t+i}\mathbb{E}[x_{t+i}]$ and $\mathbb{E}[\hat{x}_{t+i}]^{\mathsf{T}}L_t^{\mathsf{T}}R_tL_{t+i}\mathbb{E}[\hat{x}_{t+i}]$. By using ([44])

$$\frac{\partial \vec{v}^{\mathsf{T}} X \vec{w}}{\partial X} = \vec{v}\vec{w}^{\mathsf{T}}, \tag{108}$$

$$\frac{\partial \vec{v}^{\mathsf{T}} X^{\mathsf{T}} \vec{w}}{\partial X} = \vec{w}\vec{v}^{\mathsf{T}}, \tag{109}$$

$$\frac{\partial}{\partial X}(\vec{v}^{\mathsf{T}} X^{\mathsf{T}} N X \vec{v}) = 2NX\vec{v}\vec{v}^{\mathsf{T}}, \tag{110}$$

where $\vec{v}$ and $\vec{w}$ are vectors and $N$ is a symmetric matrix, we obtain

$$\mathcal{J}_t = 2R_t + 2\sum_{i=1}^{T-t}\left[V_{t+i-1}^{\mathsf{T}}(Q_{t+i} + L_{t+i}^{\mathsf{T}}R_{t+i}L_{t+i})V_{t+i-1}\right] \tag{111}$$

$$\mathcal{S}_t = 2\sum_{i=1}^{T-t}\left\{V_{t+i-1}^{\mathsf{T}}\left[Q_{t+i}\left(\mu_{L_t=0,(\mathbb{I},0)}^{t+i}\right)_1 + L_{t+i}^{\mathsf{T}}R_{t+i}L_{t+i}\left(\mu_{L_t=0,(\mathbb{I},0)}^{t+i}\right)_2\right]\right\} \tag{112}$$

$$\mathcal{P}_t = 2\sum_{i=1}^{T-t}\left\{V_{t+i-1}^{\mathsf{T}}\left[Q_{t+i}\left(\mu_{L_t=0,(0,\mathbb{I})}^{t+i}\right)_1 + L_{t+i}^{\mathsf{T}}R_{t+i}L_{t+i}\left(\mu_{L_t=0,(0,\mathbb{I})}^{t+i}\right)_2\right]\right\} \tag{113}$$

with $V_{t+i}$ given by

$$V_{t+i} = \prod_{j=1}^{i}(A + BL_{t+j})B \tag{114}$$

for $i = 1, ..., T-t$, and

$$V_t = B. \tag{115}$$

In Eqs. 112-113, $\left(\mu_{L_t=0,(\cdot,\cdot)}^{t+i}\right)$, is a vector whose elements are $m \times m$ matrices:

$$\mu_{L_t=0,(\cdot,\cdot)}^{t+i} = \begin{pmatrix}\left(\mu_{L_t=0,(\cdot,\cdot)}^{t+i}\right)_1 \\ \left(\mu_{L_t=0,(\cdot,\cdot)}^{t+i}\right)_2\end{pmatrix} \tag{116}$$

The subscript $(\cdot, \cdot)$ indicates the initial condition $(i = 0)$ for the evolution of $\mu_{L_t=0,(\cdot,\cdot)}^{t+i}$, with $\mathbb{I}$ denoting the $m \times m$ identity matrix and $0$ being an $m \times m$ matrix whose elements are all zeros, e.g.,

$$\mu_{L_t=0,(\mathbb{I},0)}^{t} = \begin{pmatrix}\mathbb{I} \\ 0\end{pmatrix}. \tag{117}$$

$\mu_{L_t=0,(\cdot,\cdot)}^{t+i}$ is updated through the following equations

$$\mu_{L_t=0,(\cdot,\cdot)}^{t+i} = \begin{cases}\tilde{M}_t\mu_{L_t=0,(\cdot,\cdot)}^{t}, & \text{for } i = 1 \\ M_{t+i-1}\mu_{L_t=0,(\cdot,\cdot)}^{t+i-1}, & \text{for } i = 2, ..., T-t\end{cases} \tag{118}$$

with $M_t$ given by Eq. 12 and $\tilde{M}_t$ having the same form as the block matrix $M_t$, but with $L_t = 0$,

$$\tilde{M}_t = \begin{pmatrix} A & 0 \\ K_t H & A - K_t H \end{pmatrix} \cdot \tag{119}$$

From Eq. 107 we can then write for Eq. 19

$$L_t^* = -\mathcal{J}_t^{-1} \left( \mathcal{S}_t \mathbb{E}[x_t \hat{x}_t^{\mathsf{T}}] + \mathcal{P}_t \mathbb{E}[\hat{x}_t \hat{x}_t^{\mathsf{T}}] \right) \mathbb{E}[\hat{x}_t \hat{x}_t^{\mathsf{T}}]^{\dagger} \tag{120}$$

where $\cdot^{\dagger}$ denotes the pseudoinverse operation. Note that $\mathcal{J}_t$ is a symmetric $p \times p$ matrix with $det[\mathcal{J}_t] > 0$ and therefore invertible. Due to the initial conditions for $\Sigma_{\hat{x}_1}$ and $\mathbb{E}[\hat{x}_1]$, the symmetric matrix $\mathbb{E}[\hat{x}_t \hat{x}_t^{\mathsf{T}}]$ could have a null determinant: for this reason we use the pseudoinverse operation. This consideration is relevant only for an initial transient: after a certain time $\tilde{t} > 0$ (depending on the dynamics parameters) we would have $det[\mathbb{E}[\hat{x}_t \hat{x}_t^{\mathsf{T}}]] > 0$ and $\mathbb{E}[\hat{x}_t \hat{x}_t^{\mathsf{T}}]^{\dagger} = \mathbb{E}[\hat{x}_t \hat{x}_t^{\mathsf{T}}]^{-1}$, due to the properties of the pseudoinverse. With Eq. 120 we can implement Algorithm 2 to find the optimal control gains $L_{1,\cdots,T-1}^*$. Notably, from the form of Eq. 120, we can see why, mathematically, the control gains decrease when the internal noise level is increased: the factor $\mathbb{E}[\hat{x}_t \hat{x}_t^{\mathsf{T}}]$ will get bigger and bigger as $\Omega_\eta$ gets larger.

The derivation of Eq. 20 for the optimal filter gains $K_t^*$ would follow the same procedure. To extend the presented approach to the case with multiplicative noise, we need to propagate the terms depending on $C_i$, $i = 1, ..., c$ and $D_i$, $i = 1, ..., d$ coming from the factors $Tr[Q_{t+i} \Sigma_{x_{t+i}}]$ and $Tr[L_{t+i}^{\mathsf{T}} R_{t+i} L_{t+i} \Sigma_{\hat{x}_{t+i}}]$ in Eq. 106, similarly to what we have done with the other terms in Eq. 118, but using ([44])

$$\frac{\partial}{\partial X} Tr[\tilde{A} X \tilde{B}] = \tilde{A}^{\mathsf{T}} \tilde{B}^{\mathsf{T}}, \tag{121}$$

$$\frac{\partial}{\partial X} Tr[\tilde{A} X^{\mathsf{T}} \tilde{B}] = \tilde{B} \tilde{A}, \tag{122}$$

$$\frac{\partial}{\partial X} Tr[\tilde{A} X \tilde{B} X^{\mathsf{T}} \tilde{C}] = \tilde{A}^{\mathsf{T}} \tilde{C}^{\mathsf{T}} X \tilde{B}^{\mathsf{T}} + \tilde{C} \tilde{A} X \tilde{B} \tag{123}$$

where $\tilde{A}$, $\tilde{B}$ and $\tilde{C}$ are matrices. We observe that, even when considering multiplicative noise, Eq. 120 will still be valid: only the matrices $\mathcal{J}_t$, $\mathcal{S}_t$ and $\mathcal{P}_t$ will change, including now also the terms depending on $C_i$, $i = 1, ..., c$, and $D_i$, $i = 1, ..., d$.

### A.6.3  Pseudo-Code

Algorithm 2 (Section 3.3) extracts the optimal solutions analytically by identifying the critical points of the total expected cost, conditioned on the first two moments of the joint variable distribution $(x_t, \hat{x}_t)$. In the pseudo-code, $L_t^{(k)}$ and $K_t^{(k)}$ stand for, respectively, the control and filter gains at time $t$ and at optimization step $k$. The hyper-parameters of the used algorithms are listed in Table 1 in Appendix A.5.

---

**Algorithm 2** FPOMP algorithm

---

    **Input:** $\mu_1$, $\Sigma_1$ (initial conditions of the system), $L_{1,\cdots,T-1}^{(1)}$, $K_{1,\cdots,T-2}^{(1)}$ (initial guesses for $L^*$ and $K^*$), and the system parameters ($A$, $B$, $H$, $C_{i=1,...,c}$, $D_{i=1,...,d}$, $\Omega_\xi$, $\Omega_\omega$, $\Omega_\eta$).
2: **Output:** $L_{1,\cdots,T-1}^*$, $K_{1,\cdots,T-2}^*$ (optimal control and filter gains)
    Algorithm steps:
4: **for** each iteration $k = 2, \dots,$ optimization steps **do**
      $\mu_{1,\cdots,T-1}, \Sigma_{1,\cdots,T-1} \leftarrow$ Eqs. 14-15 using $L_{1,\cdots,T-1}^{(k-1)}$ and $K_{1,\cdots,T-2}^{(k-1)}$
6:     **for** each iteration $i = 1, \dots, T-1$ **do**
        $t \leftarrow T - i$
8:         $L_t^{(k)} \leftarrow f(\mu_t, \Sigma_t, L_{t+1,\cdots,T-1}^{(k)}, K_{t+1,\cdots,T-2}^{(k-1)})$
        $K_t^{(k)} \leftarrow g(\mu_t, \Sigma_t, L_{t+1,\cdots,T-1}^{(k-1)}, K_{t+1,\cdots,T-2}^{(k)})$
10:    **end for**
    **end for**
12: $L_{1,\cdots,T-1}^* \leftarrow L_{1,\cdots,T-1}^{(k)}$
    $K_{1,\cdots,T-2}^* \leftarrow K_{1,\cdots,T-2}^{(k)}$

---

## A.7 Extension to Switching Linear Dynamics

We discuss here how to extend our approach to switching linear dynamics. One of the underlying assumptions in this work and in [1] is that the agent has complete knowledge of the updating rules of the latent dynamical system. By using the same set of matrices to update the state and the state estimate, we implicitly assume that all uncertainty in the estimation process arises solely from noise sources: the problem of inferring the matrices $A$ and $B$ goes beyond the objectives of this approach. For this reason, to extend our work to the more general and realistic case of Switching Linear Dynamics (SLD), we can consider a matrix $A$ depending on the time step $t$, $A_t$. A complete formulation of SLD might require adding another variable, a discrete switch variable $s_t$ regulating the way the matrices $A_t$ vary with time and context [45]. Given that in our case the agent has access to the updating rules of the dynamical system, we can omit $s_t$ (the agent does not have to infer $s_t$ and $A_t$) and directly consider the case in which we have a predetermined set of matrices $A_{1,\cdots,T-1}$. The same applies to the matrix $B$, that can be replaced by $B_{1,\cdots,T-1}$. Note that to preserve linearity we assume $A_t$ and $B_t$ to be independent on $x$ and $\hat{x}$. We consider here the multidimensional case to be as general as possible. To extend the GD algorithm we only need to modify the block matrix $M_t$ that we use to update the moments $\Sigma_t, \mu_t$ and eventually propagate the expected cost $\mathbb{E}[J]$ through Eq. 3. Indeed, once we can compute the expected cost at fixed control and filter gains, $L_{1,\cdots,T-1}$, and $K_{1,\cdots,T-2}$, we can use Algorithm 1 to define the objective function to be minimized through gradient descent with respect to $L_t$ and $K_t$. To update the block matrix $M_t$ we have to substitute $A$ and $B$ respectively with $A_t$ and $B_t$ in Eq. 12. To handle the potentially high computational costs of performing a numerical gradient descent, we introduced the analytical counterpart of the GD algorithm, the FPOMP algorithm. For the one-dimensional case, it supports all the noise sources mentioned in Section 2 (additive, multiplicative and internal). We extended this algorithm to the multi-dimensional case for additive and internal noise for the sake of simplicity, leaving the more general version for future work (Appendix A.6.2 outlines how this can be done). Here, we extend the afore-mentioned FPOMP algorithm (for both one-dimensional and multi-dimensional cases) to switching linear dynamics, following a similar procedure to that of the numerical algorithm. For the one-dimensional case, we replace $A$ and $B$ respectively with $A_t$ and $B_t$ in Eqs. 78-80, and 94-97. For the multi-dimensional case we have to substitute $A$ with $A_{t+j}$ and $B$ with $B_{t+j}$ in Eq. 114 and $B$ with $B_t$ in Eq. 115. Finally, as previously done, we replace $A$ with $A_t$ in Eq. 119 for $\tilde{M}_t$. With these changes, we can implement Algorithm 2 for the case with switching linear dynamics.

We observe that the extension to switching linear dynamics aims to make the assumption of linearity less restrictive. Additionally, given the flexibility of our approach in handling high-dimensional systems (Section 3.2), it is reasonable to think that this assumption does not limit the effective description of lower-dimensional nonlinear dynamics, potentially by employing the Koopman operator [46].

## A.8 Experiments: FPOMP Algorithm

### A.8.1 One-Dimensional Case

We show that the FPOMP and GD algorithms yield the same performance in the one-dimensional problem introduced in Section 3.2, confirming their equivalence as discussed in Section 3.2.

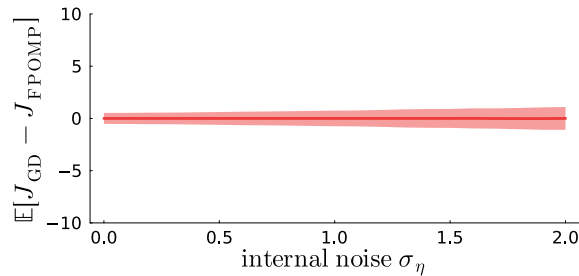

Figure 10: *Accumulated cost difference*. Difference of $\mathbb{E}[J]$ for GD and FPOMP solutions (computed by averaging the quantity from Eq. 3 over $50k$ trials), as a function of $\sigma_\eta$, with error bars (SEM).

### A.8.2  Multi-Dimensional Case

We consider the same problem described in Section 3.2 (see also Appendix A.5.4) but without multiplicative noise sources, to validate the FPOMP algorithm derived in Appendix A.6.2. The parameters of the problem are listed in Table 4.

Table 4: Parameters of the problem - sensorimotor task without multiplicative noise

| Name | Description | value |
|------|-------------|-------|
| $\Delta t$ | time-step $(s)$ | 0.010 |
| $m$ | mass of the hand $(Kg)$, modelled as a point mass | 1 |
| $\tau_1$ | time constant of the second order low pass filter | 0.04 |
| $\tau_2$ | time constant of the second order low pass filter | 0.04 |
| $r$ | Control-dependent cost at each $t < T$ | $1e^{-5}$ |
| $w_v$ | Task-related cost at time $t = T$ for the velocity | 0.2 |
| $w_f$ | Task-related cost at time $t = T$ for the force | 0.01 |
| $T$ | time steps | 50 |
| $z$ | Target position | 0.15 |
| $\sigma_z$ | Target position standard deviation | 0.0 |
| $\sigma_\xi$ | std of the additive Gaussian noise $\xi_t$ | 0.5 |
| $\sigma_\omega$ | std of the additive Gaussian noise $\omega_t$ | 0.5 |
| $\sigma_\varepsilon$ | std of the control-dependent noise $\varepsilon_t$ | 0.0 |
| $\sigma_\rho$ | std of the signal-dependent noise $\rho_t$ | 0.0 |
| $\sigma_\eta$ | std of the additive internal noise $\eta_t$ for the position estimate | $\{0.0, 1.0, 2.0\}$ |
| $\sigma_{\eta_v}$ | std of the additive internal noise $\eta_t$ acting on the velocity estimate | 0 |
| $\sigma_{\eta_f}$ | std of the additive internal noise $\eta_t$ for the force estimate | 0 |
| $\sigma_{\eta_g}$ | std of the additive internal noise $\eta_t$ for the estimate of $g$ | 0 |

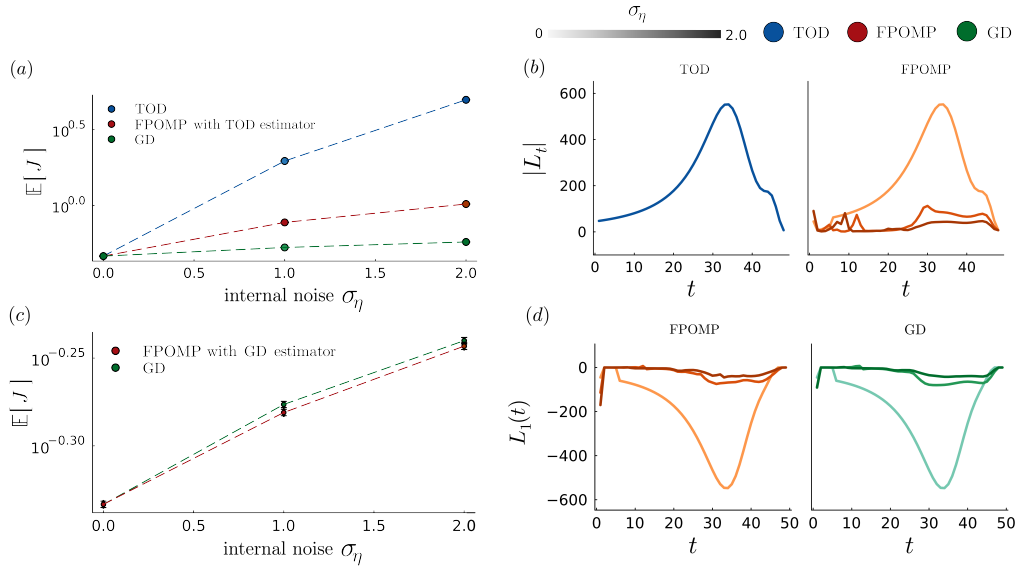

Figure 11: *Application of the FPOMP algorithm.* **(a)** $\mathbb{E}[J]$, computed by averaging the quantity from Eq. 3 over $50k$ trials, as a function of $\sigma_\eta$, with error bars (mean $\pm$ 1SEM from Monte Carlo simulations, error bars not visible as too small) and for TOD, GD and FPOMP (at fixed filters given by TOD solution). **(b)** Magnitude of the control gain vector as a function of time for TOD and FPOMP (at fixed filters given by TOD). **(c)** Same as **(a)**, but comparing GD and FPOMP (now at fixed filters given by GD). **(d)** First component of the vector $L_t$ for the solution given by GD and FPOMP (now at fixed filters given by GD solution).

We recall that, for readability, we derived solutions only for the optimal controller; the procedure for the optimal estimator follows similarly but was not explicitly derived. Extensions to the optimal estimator and the general case with multiplicative noise are left for future work, as discussed in Appendix A.6.2.

Optimizing the control gains $L_t$ using the FPOMP algorithm (with fixed filter gains $K_t$ from the TOD solution) leads to improved performance (Fig. 11a, orange dashed line) as internal noise increases. However, this solution is not fully optimal, as the estimator is still optimized using the TOD algorithm. When $L_t$ and $K_t$ are both optimized with the GD method, a lower accumulated cost is achieved (green dashed line). An interesting feature of our algorithm is that, being fully analytical, it can enhance numerical solutions. Due to a potentially shallow parameter landscape (vanishing gradient) or limited computation time, the GD optimization may stop near the global optimum without fully reaching it. We find that re-optimizing the control gains $L_t$ using the FPOMP algorithm after the GD solution for filter gains yields a small but significant performance boost (Fig. 11c), with minimal changes in the final $L_t$ vector (Fig. 11d). This also confirms that our algorithm finds the optimal solutions. Extensions to estimator optimization and the multiplicative noise case are discussed in Appendix A.6.2.

The qualitative trends observed in the sensorimotor task (Section 3.2, Fig. 3) are confirmed: control magnitude decreases as internal noise increases (Fig. 11b). Additionally, while the TOD solution does not modulate control with respect to internal noise when only additive noise is present, the FPOMP algorithm introduces such modulation, leading to a lower accumulated cost (Figs. 11a,b), consistent with our discussion in Appendix A.5.2.

